# Addressing Spatial-Temporal Heterogeneity: General Mixed Time Series Analysis via Latent Continuity Recovery and Alignment

**Jiawei Chen, Chunhui Zhao**[*]
State Key Laboratory of Industrial Control Technology,
College of Control Science and Engineering, Zhejiang University, China

## Abstract

Mixed time series (MiTS) comprising both continuous variables (CVs) and discrete variables (DVs) are frequently encountered yet under-explored in time series analysis. Essentially, CVs and DVs exhibit different temporal patterns and distribution types. Overlooking these heterogeneities would lead to insufficient and imbalanced representation learning, bringing biased results. This paper addresses the problem with two insights: 1) DVs may originate from intrinsic latent continuous variables (LCVs), which lose fine-grained information due to extrinsic discretization; 2) LCVs and CVs share similar temporal patterns and interact spatially. Considering these similarities and interactions, we propose a general MiTS analysis framework **MiTSformer**, which recovers LCVs behind DVs for sufficient and balanced spatial-temporal modeling by designing two essential inductive biases: 1) hierarchically aggregating multi-scale temporal context information to enrich the information granularity of DVs; 2) adaptively learning the aggregation processes via the adversarial guidance from CVs. Subsequently, MiTSformer captures complete spatial-temporal dependencies within and across LCVs and CVs via cascaded self- and cross-attention blocks. Empirically, MiTSformer achieves consistent SOTA on five mixed time series analysis tasks, including classification, extrinsic regression, anomaly detection, imputation, and long-term forecasting. The code is available at `https://github.com/chunhuiz/MiTSformer`.

## 1 Introduction

Multivariate time series analysis is energized in various real-world applications, such as weather forecasting [7], activity recognition [20], and industrial maintenance [47]. Empowered by deep learning, plentiful time series models have been proposed based on foundation models such as RNNs [23, 25, 22], CNNs [6, 36, 40], Transformers [30, 41, 10] and modernized MLPs [50, 45, 39]. These sophisticated models have achieved increasingly remarkable performance in various time series analysis tasks, e.g., classification [17], forecasting [7, 42], imputation [40] and anomaly detection [9].

The primary challenge in time series analysis is effectively modeling spatial-temporal patterns, including intra-variable temporal variations and inter-variable spatial correlations [46, 48, 16, 11, 43]. So far, most current approaches naturally assume that time series data are composed solely of continuous variables, and then uniformly model spatial-temporal patterns in continuous spaces. Yet, in broad practical scenarios, the acquired data are often mixed time series (MiTS) that encompass both continuous variables (CVs) and discrete variables (DVs). Take meteorological data as an example (Fig. 1 (Left)), some sensor-derived variables are commonly recorded as CVs (e.g., temperature,

---

[*]Corresponding author: Chunhui Zhao (`chhzhao@zju.edu.cn`)

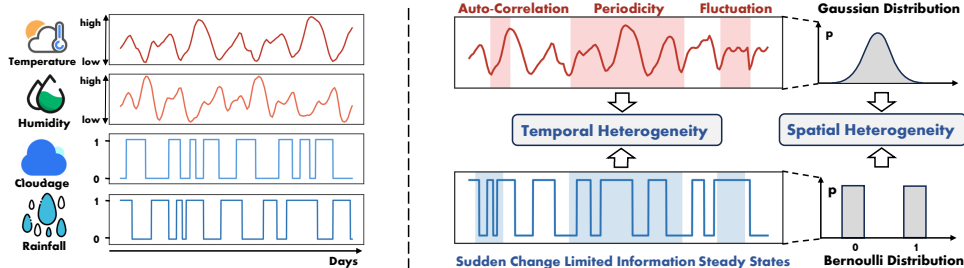

Figure 1: *Left*: Illustration of mixed time series. *Right*: Spatial-temporal heterogeneity problem.

humidity, and wind speed), while certain variables like cloudage and rainfall patterns are typically tracked as DVs due to measurement restrictions or distinctive nature. Up to now, mixed time series analysis is still a formidable yet under-explored problem. Essentially, MiTS presents spatial-temporal heterogeneity problems as depicted in Fig. 1 (Right). On one hand, CVs commonly encapsulate rich temporal variation information, exhibited in *autocorrelations, periodical patterns, local fluctuations*, etc, while DVs often undergo *sudden changes* or *steady states* due to limited value ranges, resulting in the temporal variation discrepancy between CVs and DVs that complicates temporal modeling. On the other hand, CVs generally adhere to Gaussian distributions, while DVs follow Bernoulli distributions, resulting in the distribution type discrepancy that hinders spatial correlation analysis between CVs and DVs. Neglecting these heterogeneities and equally treating mixed variables would yield insufficient and imbalanced spatial-temporal modeling problems, i.e., the model may struggle to characterize distinct temporal patterns of DVs and CVs and fail to reliably capture spatial correlations within and across DVs and CVs, posing a bottleneck for MiTS analysis.

The key to addressing the spatial-temporal heterogeneity lies in bridging the information gap between DVs and CVs. Essentially, in real-world MiTS, the observed DVs may originate from intrinsic continuous-valued factors, which are unobservable owing to extrinsic factors such as measurement limitations, storage requirements, and transmission interference. Continuing with the aforementioned meteorological example, cloud cover percentage is an intrinsically continuous variable, whose fine-grained values are hard to measure directly. In practice, discretizing it with coarse-grained discrete variable-cloudage (reflect "cloudy" or not) is sufficient for most applications and is memory-efficient. In this paper, we introduce latent continuous variables (LCVs) to describe the intrinsic continuous factors behind DVs. Given its numerically continuous nature, the LCV of cloudage, i.e., cloud cover percentage, may be spatially correlated with other observed CVs (e.g., humidity and wind speed) and exhibit similar temporal variation patterns (e.g., autocorrelation and seasonal fluctuation) with them, as both of them originate from the same meteorological system. Thereby, we can progressively recover the LCVs behind DVs by leveraging these temporal similarities and spatial interactions among CVs and LCVs. In this way, spatial-temporal dependencies of mixed variables can be completely and reliably modeled in a unified continuous numerical space, and the spatial-temporal heterogeneity problem is mitigated. Also, by bridging the mutual spatial-temporal interactions, LCVs and CVs can supply complementary information for various downstream analysis tasks.

Enlightened by the above insights, we reconcile the intrinsic tension between the two highly dependent problems - Latent Continuity Recovery and Spatial-Temporal Modeling - in one coherent and synergistic framework, **MiTSformer**, for general mixed time series analysis. By leveraging the temporal similarities and spatial interactions between LCVs and CVs, MiTSformer can gradually decipher the LCVs behind DVs and capture the complete spatial-temporal dependencies within and across LCVs and CVs. Specifically, we design a recovery network to portray LCVs behind DVs by adaptively and hierarchically aggregating temporal contextual information. Following, an adversarial variable modality discrimination objective and smoothing constraints are devised to guide the learning of the recovery network, ensuring the recovered LCVs share similar temporal properties and distributions with CVs. Additionally, MiTSformer employs self-attention to learn spatial-temporal dependencies within LCVs or CVs and cross-attention to exploit those across LCVs and CVs, facilitating various downstream analysis tasks. Our contributions lie in three aspects:

(1) **Novel Problem**: To the best of our knowledge, our paper pioneers the exploration of the general mixed time series analysis, which is practical and challenging. We reveal the crucial spatial-temporal heterogeneity problem, which is caused by the discrepancies in temporal variation properties and distribution types between DVs and CVs.

(2) **Customized Framework**: To solve the spatial-temporal heterogeneity problem, we propose a task-general framework MiTSformer customized for MiTS, which adaptively recovers LCVs behind DVs by leveraging the adversarial guidance of CVs and task supervisions. Moreover, MiTSformer can capture spatial-temporal dependencies within and across CVs and LCVs via self- and cross-attention blocks, thus learning sufficient and balanced spatial-temporal representations and being amenable to various mixed time series analysis tasks.

(3) **Versatile Effectiveness**: Empirically, our proposed MiTSformer establishes the state-of-the-art performance on five mainstream mixed time series analysis tasks with 34 datasets covering wide-ranging real-world application domains. We believe our work makes a predominant attempt at general mixed time series analysis in practical applications.

## 2 Related Work

**General Time Series Analysis** General time series analysis aim at learning universal temporal representations by developing task-general backbones and task-specific heads for diverse tasks. To date, these methods have been designed predominantly for time series comprising continuous variables solely. As a pioneering work, TimesNet [40] leverages fast Fourier transform and parameter-efficient inception block to capture intra-period and inter-period variations for time series modeling. Followingly, ModernTCN [16] modernizes and modifies the traditional TCN by introducing larger effective receptive fields and cross-variable dependency modeling, bringing great performance and efficiency. Meanwhile, GPT2TS [52] leverages pre-trained language models, e.g., GPT2 [32], for various time series analysis tasks with task-specific fine-tuning.

Yet, most of the current works naturally assume uniformity in variable types of time series and are inapplicable for MiTS, as they lack the differentiation of distinct variable modalities. In this paper, we develop a systematic framework with delicate differentiation and deft alignment of mixed variables to support sufficient and balanced spatial-temporal modeling for general mixed time series analysis.

**Mixed Data Analysis** Previous studies have revealed the significance of mixed data analysis and made several attempts to shed light on this challenging problem. A naive solution for mixed data modeling is roughly pre-processing DVs and CVs into the same variable modality, e.g., discarding DVs or discretizing CVs by certain policies [35, 13]. However, these methods may lose vital fine-scaled information and bring errors inevitably. Towards concurrently modeling of DVs and CVs, Mixed Data RBF-ELM method [24] adopts a distance-based learning scheme for efficient and direct mixed data classification, while Mixed-variate Restricted Boltzmann Machine (Mv.RBM) [15, 14] construct ensembles of mixed-data Deep Belief Nets with varying depths for anomaly detection of mixed data. Another line of work treats DVs as semantic attributes of CVs and establishes the relationships between DVs and CVs by developing specific inference rules [18, 44]. More recently, researchers have utilized mixed naive Bayes models [37, 38] or variational inference [12] for mixed data modeling in industrial processes with different distribution priors of CVs and DVs.

However, current studies mostly focus on specific analysis tasks (e.g., designed only for classification) and may be restricted by linear, non-temporal data, or other rigorous assumptions, which are not capable of handling real-world MiTS that exhibit intricate spatial-temporal patterns. Instead, our proposed MiTSformer can model complete spatial-temporal dependencies within and across DVs and CVs and can produce task-general representations for various MiTS analysis tasks.

## 3 MiTSformer

### 3.1 Problem Formulation and Motivation Analysis

**Mixed Time Series**. Given a collection of multivariate time series $X = \{x_1, x_2, ..., x_p\}$ comprising $p$ variables with length $T$. Among them, there are $p - n$ continuous variables $X^C = \{x_1, x_2, ..., x_{p-n}\} \in \mathbb{R}^{(p-n) \times T}$ with continuous numerical values, and $n$ discrete variables $X^D = \{x_{p-n+1}, x_{p-n+2}, ..., x_n\} \in \mathbb{A}^{n \times T}$ with discrete states. Without loss of generality, we consider the binary-valued discrete variables, whose value can be 0 or 1, i.e., $\mathbb{A} = \{0, 1\}$. Mixed time series are used as model input to support various analysis tasks, such as regression and classification.

Hampered by the spatial-temporal heterogeneity problem, directly modeling spatial-temporal dependencies of CVs and DVs without considering their discrepancies may inevitably yield non-negligible

biases. Such research bottlenecks prompt us to exploit the underlying generation and interaction mechanism of DVs and CVs. As aforementioned, the observed DVs are potentially derived from LCVs, which undergo discretization processes due to external interferences as depicted in Fig. 2. For each DV $x^D$, we adopt a corresponding LCV $x^{LC}$ to portray its latent continuity. To reliably and completely model inherent spatial-temporal patterns within MiTS, deciphering and recovering LCVs behind DVs is indispensable. Yet, it is challenging since a single discrete state can not be directly transformed into a continuous value without proper supervision. In this study, we address this challenge by revealing and leveraging the temporal similarity and spatial interaction between LCVs and CVs:

**Temporal Similarity** The unobserved LCVs share similar temporal variation patterns (e.g., autocorrelation, periodicity, trend, etc.) with the observed CVs.

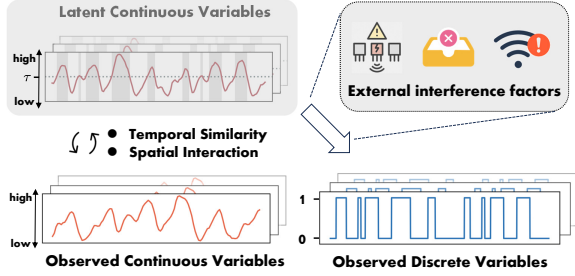

**Spatial Interaction** LCVs and CVs exhibit information interactions and inter-variable spatial correlations. The synergistic effect of LCVs and CVs provides complementary information for various downstream tasks.

Figure 2: Connections among DVs, CVs, and LCVs.

Built upon these two insights, we design MiTSformer, which will be elaborately introduced in the following parts.

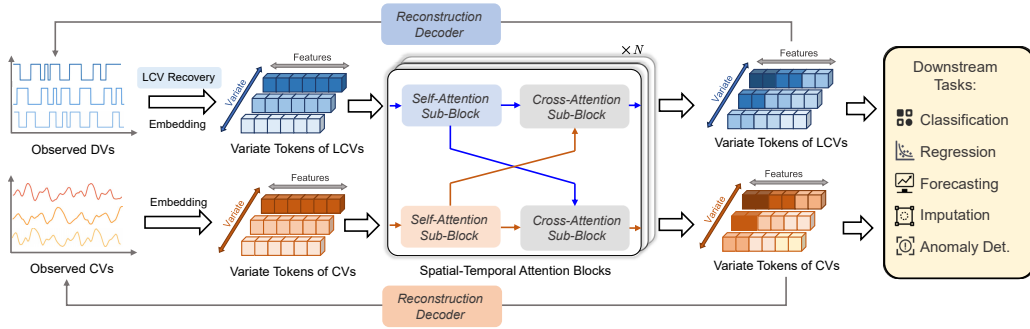

Figure 3: Overall pipeline of MiTSformer. First, MiTS undergo latent continuity recovery (DVs only) and are embedded as variate tokens, which are then refined through spatial-temporal attention blocks. The acquired variate tokens are utilized both for reconstructing the original MiTS and serving various downstream tasks.

## 3.2 Framework Overview

As aforementioned, there are two key steps for general MiTS analysis: 1) unifying the temporal characteristics and distribution types of DVs and CVs; 2) modeling sufficient and balanced spatial-temporal dependencies for effective representation learning. MiTSformer facilitates these two steps in a highly versatile manner with a coherent framework as depicted in Fig. 3. The overall pipeline contains two key parts: 1) *Latent Continuity Recovery* that adaptively aggregate contextual information of DVs to recover LCVs with the adversarial alignment guidance of CVs and temporal smoothness constraints; 2) *Spatial-Temporal Attention Blocks* that model intra- and inter-variable modality spatial-temporal dependencies with cascaded self-attention and cross-attention sub-blocks.

## 3.3 Latent Continuity Recovery

Since single time points of DVs contain limited information, it is difficult to directly transform discrete states to latent continuous values. Fortunately, time series commonly present auto-correlation natures, i.e., the value of a certain time step is correlated with its adjacent ones. Specifically, for a discrete state of "1" surrounded by states "1", its latent continuous value would be relatively large, e.g., 0.9. In contrast, for a discrete state of "1" surrounded by states "0", its latent continuous value would be relatively small, e.g., 0.6. Thus, an intuitive solution for enriching the information density of a single time point of DVs is to properly leverage its adjacent context information.

**Contextual Aggregation Recovery Network** We convert the above insights into the model inductive bias and realize the latent continuity recovery by *adaptively and hierarchically aggregating multi-scale adjacent context information* of DVs. Specifically, convolutional neural networks (CNNs) own an inductive bias that can aggregate receptive local information by convolutional kernels. Technically, we devise the recovery networks that receive DVs as input to generate LCVs as $x^{LC} = \text{Rec-Net}\left(x^D\right)$. As depicted in Fig. 4, the recovery network is composed of several residual dilated convolutional blocks, which employ dilated convolutional kernels along the temporal dimension to aggregate adjacent context information, and utilize residual connections to adaptively accumulate multi-scale temporal information to characterize intricate temporal patterns of LCVs. The residual dilated convolutional network is implemented by the iterative process as $h_i = \text{Conv}^{d_i}(h_{i-1}) + h_{i-1}$, where $h_i$ represents the output of the $i$-th residual block, $i = 1, 2, \ldots, n$, $\text{Conv}^{d_i}$ denotes the convolution operation along the temporal axis to aggregate contextual information with dilation rate $d_i$. The final output, $x^{LCV} = h_n$, is the result after $n$ residual blocks.

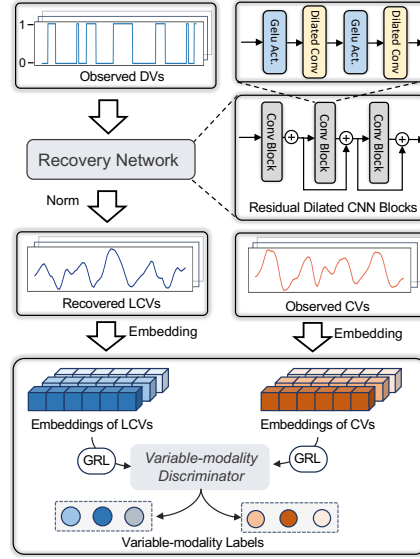

Figure 4: LCV recovery with adversarial variable modality discrimination.

**Temporal Adjacent Smoothness Constraint**: To facilitate spatial-temporal modeling across LCVs and CVs, the recovered LCVs should be equipped with interpretable autocorrelation or trend properties, instead of "sudden changes" as DVs. To this end, we encourage the smoothness of LCVs across time with a regularization term as

$$\mathcal{L}_{\text{smooth}} = \left\|\text{Abs}\left(\boldsymbol{S}x^D\right) \otimes \left(\boldsymbol{S}x^{LC}\right)\right\|_2^2, \boldsymbol{S} = \begin{bmatrix} -1 & 1 & & \\ & \ddots & \ddots & \\ & & -1 & 1 \end{bmatrix} \in \mathbb{R}^{(T-1)\times T} \tag{1}$$

where $\boldsymbol{S}$ is the smoothness matrix and $\otimes$ denotes the Hadamard product operation. $\text{Abs}\left(\boldsymbol{S}x^D\right) = \text{Abs}\left(x^D_{[2:T]} - x^D_{[1:T-1]}\right)$ denotes the absolute value of the first-order difference of DVs and can reflect the "*sudden change*" points (with state "1"). Overall, minimizing $\mathcal{L}_{\text{smooth}}$ is equivalent to minimizing $\sum_{t=1}^{T-1} \text{Abs}\left(x^D_{t+1} - x^D_t\right)\left(x^{LC}_{t+1} - x^{LC}_t\right)^2$ by introducing the multiplication of the constant matrix $\boldsymbol{S}$.

Alternatively, we can adopt the K-Lipschitz continuity as smoothness constraints, where the CVs can act as guidance to determine the smoothness degree of the recovered LCVs. Empirically, we find out such design would bring similar performance.

After latent continuity recovery, raw series of $x^{LC}$ and $x^C$ are independently embedded as tokens $\boldsymbol{z}^{LC}$ and $\boldsymbol{z}^C$ through variate-wise linear projection to describe the properties of each variable.

**Adversarial Variable-Modality Discrimination**: Inspired by the temporal similarity property, the recovered LCVs should exhibit similar temporal patterns and distributions to CVs. Accordingly, we devise a variable modality discrimination objective, which is optimized in an adversarial manner as

$$\underset{\theta_{\text{Dis}}}{\arg\min}\left(\underset{\theta_{\text{Rec}},\theta_{\text{Emb}}}{\max}\left(\mathcal{L}_{\text{Dis}} = \mathbb{E}\left[\log\left(\text{Dis}\left(\boldsymbol{z}^C\right)\right)\right] + \mathbb{E}\left[\log\left(1 - \text{Dis}\left(\boldsymbol{z}^{LC}\right)\right)\right]\right)\right) \tag{2}$$

where $\text{Dis}$ denotes the variable modality discriminator. $\text{Dis}$ is trained to distinguish LCVs and CVs as accurately as possible, while the recovery networks are facilitated to generate LCVs with characteristics similar to CV as much as possible to confuse the discriminator. We adopt gradient reversal layer [19](GRL) to achieve the adversarial learning objective. In this way, we bridge the interactions between CVs and DVs and enable CVs to supply supervision signals for LCV recovery.

### 3.4 Intra- and Inter-Variable-Modality Spatial-Temporal Dependency Learning

Spatial-temporal dependencies are of vital importance for time series representation learning. Specifically for MiTS, the complete spatial-temporal correlations include the ones within LCVs or CVs

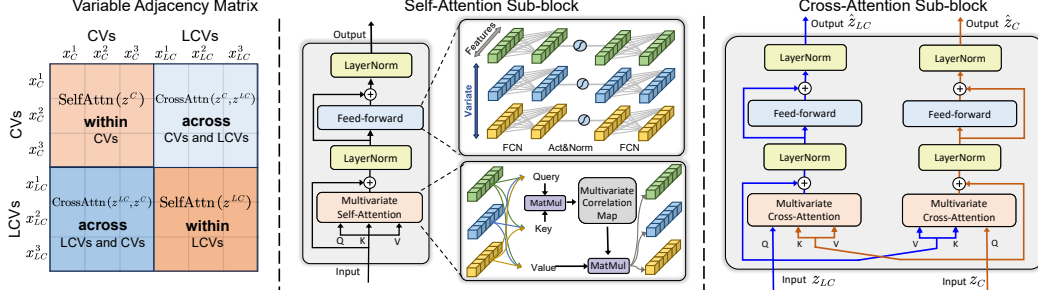

Figure 5: Spatial-temporal attention blocks. *Left*: MiTS variable adjacency matrix, including the variable relationships i) *within* CVs or LCVs and ii) *across* CVs and LCVs; *Middle*: Intra-variable-modality self-attention for modeling spatial-temporal dependencies *within* CVs or LCVs, and *Right*: Inter-variable-modality cross-attention for modeling those *across* CVs and LCVs.

and the ones across LCVs and CVs, which are explicitly characterized by spatial-temporal attention blocks in MiTSformer (as shown in Fig. 5).

**Intra-Variable-Modality Self-Attention** Within each variable modality, we adopt self-attention to model their spatial-temporal dependencies as

$$\hat{z}_l^C = \text{LN}\left(z_l^C + \text{Self-Attn}\left(\left[Q_l^C, K_l^C, V_l^C\right]\right)\right), \, \hat{z}_l^C = \text{LN}\left(\hat{z}_l^C + \text{FFN}\left(\hat{z}_l^C\right)\right)$$

$$\hat{z}_l^{LC} = \text{LN}\left(z_l^{LC} + \text{Self-Attn}\left(\left[Q_l^{LC}, K_l^{LC}, V_l^{LC}\right]\right)\right), \, \hat{z}_l^{LC} = \text{LN}\left(\hat{z}_l^{LC} + \text{FFN}\left(\hat{z}_l^{LC}\right)\right) \quad (3)$$

where $\text{Self-Attn}$ denotes the Multi-head Self-Attention that captures intra-variate spatial correlations by computing Softmax scores with query and key embeddings and weighted aggregating the value embeddings. FFN denotes the feed-forward network that processes each variable token to learn intra-variate global temporal representations. LN denotes Layer Normalization, which is applied to individual variate tokens and has been proven effective in tackling non-stationary problems [28].

**Inter-Variable-Modality Cross-Attention** We adopt symmetric cross-attention sub-blocks to model the spatial-temporal interactions across LCVs and CVs as

$$z_{l+1}^C = \text{LN}\left(\hat{z}_l^C + \text{Cross-Attn}\left(\left[Q_l^C, K_l^{LC}, V_l^{LC}\right]\right)\right), \, z_{l+1}^C = \text{LN}\left(z_{l+1}^C + \text{FFN}\left(z_{l+1}^C\right)\right)$$

$$z_{l+1}^{LC} = \text{LN}\left(z_{l+1}^{LC} + \text{Cross-Attn}\left(\left[Q_l^{LC}, K_l^C, V_l^C\right]\right)\right), \, z_{l+1}^{LC} = \text{LN}\left(z_{l+1}^{LC} + \text{FFN}\left(z_{l+1}^{LC}\right)\right) \quad (4)$$

Subsequently, the acquired token embeddings $z_L^{LC}$ and $z_L^C$ are utilized both for 1) original MiTS reconstruction with $\mathcal{L}_{\text{Rec}}$ as Eq. 5, and 2) downstream tasks with task supervision loss $\mathcal{L}_{\text{Task}}$, e.g., Cross Entropy loss for the classification task. See Appendix A for pipelines of each task.

**Self-Reconstruction** We devise variate-wise decoders based on MLPs to reconstruct the original DVs and CVs, which can not only provide self-supervision signals for spatial-temporal dependency learning but also guarantee the recovered LCVs retain the information of the observed DVs.

$$\mathcal{L}_{\text{Rec}} = \sum_{i=1}^{p-n} \text{MSE}(\text{Rec-Decoder}_C\left(z_{L,i}^C\right), x_i^C) + \sum_{i=1}^{n} \text{CE}(\text{Rec-Decoder}_{LC}\left(z_{L,i}^{LC}\right), x_i^D) \quad (5)$$

### 3.5 Synergy of Latent Continuity Recovery and Downstream Tasks

Considering both latent continuity recovery and downstream analysis task, the overall optimization objective of MiTSformer is expressed as

$$\mathcal{L}_{\text{All}} = \underbrace{\lambda_1 \mathcal{L}_{\text{Smooth}} + \lambda_2 \mathcal{L}_{\text{Rec}} + \lambda_3 \mathcal{L}_{\text{Dis}}}_{\text{Self-Supervision}} + \underbrace{\mathcal{L}_{\text{Task}}}_{\text{Task-Supervision}} \quad (6)$$

While seemingly separated from each other, the loss components of MiTSformer work in a collaborative fashion on two aspects: 1). **The first three self-supervision losses facilitate latent continuity recovery synergetically** to tackle the spatial-temporal heterogeneity problem. Specifically, the smoothness loss $\mathcal{L}_{\text{Smooth}}$ favorably manifests the autocorrelation of LCVs and alleviates the sudden

Table 1: Summary of experiment benchmarks. For each dataset, we randomly select $n = \lfloor 0.5p \rfloor$ variables as DVs, whose values are first MinMax normalized and then discretized into the value of 0 or 1 with the threshold 0.5 as $\texttt{int}(\texttt{MinMax}(x) > 0.5)$. See Table 5 for more details.

| Tasks | Benchmarks | Metrics | Series-length | #Variables ($p$) |
|---|---|---|---|---|
| Classification | UEA (10 subsets) | Accuracy | 29∼1751 | 3∼963 |
| Extrinsic Regression | UCR (10 subsets) | MAE,RMSE | 24∼1140 | 4∼24 |
| Imputation | ETT (4 subsets), Electricity,Weather | MSE, MAE | 96 | 7∼321 |
| Anomaly Detection | SMD, MSL, SMAP, SWaT, PSM | Precision, Recall, F1-Socre | 100 | 25∼55 |
| Long-term Forecasting | ETT (4 subsets), Electricity, Traffic, Weather, Exchange, ILI | MSE, MAE | 96∼720 (ILI: 24∼60) | 7 ∼ 862 |

changes. Also, the discrimination loss $\mathcal{L}_{\text{Dis}}$ guarantees LCVs to be equipped with similar temporal dynamics with CVs adversarially, which guarantees a crucial condition for spatial-temporal modeling. Meanwhile, the reconstruction loss provides auxiliary self-supervision signals and constrains the LCV recovery processes to be reversible. Our experiments, particularly the ablation study displayed in Section 4.2, further justify the mutual dependency of the synergy between the three components; 2) **The latent continuity recovery losses and task loss also work collaboratively**, as reliable latent continuity recovery can bring excellent downstream task performance, while task supervision loss may, in turn, also reciprocate the recovery processes. Take MiTS classification as an example, the ideally recovered LCVs can prompt learning discriminative representations for classification, while the class label loss provides additional supervision for learning appropriate recovery functions.

# 4 Experiments

To verify the effectiveness and versatility of MiTSformer, we extensively experiment on five mainstream mixed time series analysis tasks, including *mixed time series classification, extrinsic regression, long-term forecasting, imputation, and anomaly detection*.

**Implementations** Table 1 summarizes the experiment benchmarks. For each dataset, we randomly select *half the variables* and discretize them as DVs to generate MiTS data. More information about datasets and experimental platforms, hyperparameters and experimental configurations, and algorithm implementations can be found in Appendix A.1, A.2,A.3, respectively. The pipelines of different mixed time series analysis tasks can be found in Appendix A.5∼A.8.

**Baselines** We extensively compare MiTSformer with the latest and advanced models in the time series community, including CNN-based models: ModernTCN (2024), TimesNet (2023) and MICN (2023); Transformer-based models: iTransformer (2024), PatchTST (2023), Crossformer (2023), FEDformer (2022) and Pyraformer (2022); MLP-based models: LightTS (2023), DLinear (2023) and FiLM (2022). To guarantee fairness, we keep the original backbone for each method as the feature extractor, and we adopt universal task-specific heads and loss functions consistently for all methods.

## 4.1 Main Results on Different Tasks

**(1) Mixed Time Series Classification** We select 10 multivariate datasets from the UEA Time Series Classification Archive [5] and pre-press them following [40]. As shown in Fig. 6, MiTSformer achieves the best performance with an average accuracy of 71.9%, surpassing all powerful baselines. Besides, it can be observed that frequency-based methods FiLM and FEDformer show inferior performance, as the introduction of DVs may make it difficult to estimate the frequency reliably and yield non-negligible errors. In comparison, MiTSformer adaptively recovers and aligns the LCVs behind DVs with the guidance of both CVs and class-label supervision, thereby facilitating high-level representation learning and benefiting the classification tasks.

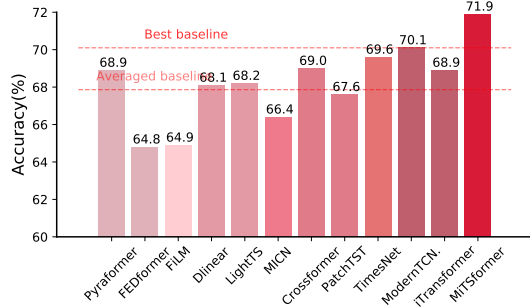

Figure 6: Classification Results (Acc ↑)

Table 2: Imputation Task. The best results are **bolded** and the second-best results are underlined. The same goes for Table 3. See Table 14 for full results.

| Models | MiTSformer (Ours) | | iTrans. (2024) | | M-TCN. (2024) | | TimesNet (2023) | | PatchTST (2023) | | Cross. (2023) | | MICN (2023) | | LightTS (2023) | | Dlinear (2023) | | FiLM (2022) | | FED. (2022) | | Pyra. (2022) | |
|---|---|---|---|---|---|---|---|---|---|---|---|---|---|---|---|---|---|---|---|---|---|---|---|---|
| Metric | MAE | MSE | MAE | MSE | MAE | MSE | MAE | MSE | MAE | MSE | MAE | MSE | MAE | MSE | MAE | MSE | MAE | MSE | MAE | MSE | MAE | MSE | MAE | MSE |
| ETTm1 | 0.156 | 0.049 | 0.169 | 0.057 | **0.135** | **0.037** | 0.139 | 0.039 | 0.164 | 0.057 | 0.160 | 0.051 | 0.160 | 0.052 | 0.178 | 0.064 | 0.193 | 0.080 | 0.194 | 0.080 | 0.170 | 0.060 | 0.199 | 0.081 |
| ETTm2 | **0.116** | **0.036** | 0.186 | 0.080 | 0.188 | 0.077 | 0.170 | 0.065 | 0.145 | 0.055 | 0.216 | 0.103 | 0.245 | 0.131 | 0.209 | 0.095 | 0.253 | 0.147 | 0.258 | 0.152 | 0.238 | 0.123 | 0.265 | 0.141 |
| ETTh1 | 0.223 | 0.096 | 0.241 | 0.116 | **0.215** | **0.092** | 0.237 | 0.112 | 0.251 | 0.129 | 0.246 | 0.121 | 0.234 | 0.107 | 0.266 | 0.145 | 0.262 | 0.145 | 0.268 | 0.152 | 0.244 | 0.112 | 0.247 | 0.113 |
| ETTh2 | **0.186** | **0.083** | 0.250 | 0.134 | 0.309 | 0.213 | 0.319 | 0.239 | 0.260 | 0.148 | 0.304 | 0.198 | 0.318 | 0.220 | 0.314 | 0.209 | 0.312 | 0.211 | 0.342 | 0.262 | 0.350 | 0.258 | 0.367 | 0.253 |
| Electric | **0.186** | **0.076** | 0.212 | 0.093 | 0.207 | 0.088 | 0.211 | 0.094 | 0.203 | 0.115 | 0.209 | 0.097 | 0.229 | 0.103 | 0.223 | 0.099 | 0.257 | 0.130 | 0.257 | 0.128 | 0.260 | 0.130 | 0.274 | 0.150 |
| Weather | **0.062** | **0.031** | 0.091 | 0.038 | 0.081 | 0.033 | 0.149 | 0.065 | 0.088 | 0.037 | 0.115 | 0.043 | 0.135 | 0.055 | 0.107 | 0.041 | 0.121 | 0.048 | 0.124 | 0.184 | 0.139 | 0.057 | 0.100 | 0.042 |

**(2) Mixed Time Series Extrinsic Regression** We select 10 multivariate datasets from TSER repository [34] and pre-process them as [34]. As summarized in Fig. 7 (b), MiTSformer outperforms all the rivals with an average MAE of 0.524, verifying its capacity to model complex extrinsic regression patterns. Besides, some transformer-based models, e.g., PatchTST and Crossformer, present competitive performance by learning global dependencies with self-attention. Notably, Dlinear shows inferior results. This is probably because Dlinear adopts single-layer MLP to model temporal dependencies, which might be suitable for some autoregressive endogenous patterns, but are not applicable for nonlinear exogenous regression relationships and degenerate performance.

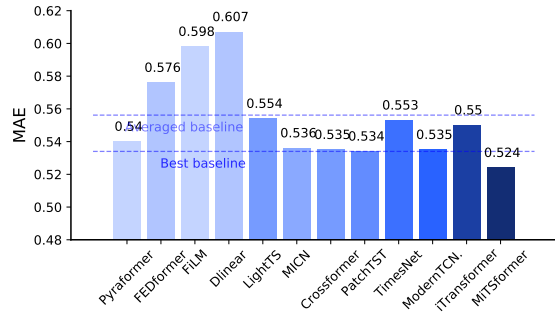

Figure 7: Regression Results (MAE ↓)

**(3) Mixed Time Series Anomaly Detection** We conduct experiments on five widely-used anomaly detection benchmarks: SMD [33], MSL [21], SMAP [21], SWaT [29] and PSM [4].

For fair comparisons, we adopt the reconstruction loss for both CVs and DVs to train base models and use the reconstruction error as the shared anomaly criterion for all experiments. Specifically for MiTSformer, no additional task-orientation loss is added since there are already reconstruction losses. Fig. 8 presents the performance comparison evaluated by the F1-score (↑), demonstrating that MiTSformer consistently achieves state-of-the-art performance on five benchmarks. Besides, iTrasformer also achieves great performance by adaptively modeling multivariate correlations with self-attentions. In comparison, MiTSformer not only recovers the LCVs behind DVs but also comprehensively and explicitly characterizes the spatial-temporal dependencies within and across DVs and CVs, thereby equipping meticulous anomaly detection ability.

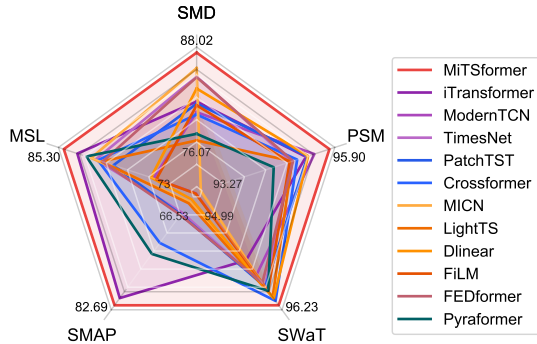

Figure 8: Anomaly detection results (F1-score).

**(4) Mixed Time Series Imputation** Following [40], we select datasets of ETT [49], Weather (Wetterstation) and Electricity (UCI) benchmarks. In practice, imputating the values of CVs is more important and meaningful than those of DVs. Therefore, we focus on imputing the CVs in each experiment, while DVs are used as inputs to provide auxiliary information. To compare the model capacity under different proportions of missing data, we randomly mask the time points in the ratio of $\{12.5\%, 25\%, 37.5\%, 50\%\}$ in length-96 mixed time series and report the averaged imputation accuracy of CVs from 4 different mask ratios.

Table 3: Long Term Forecasting of CVs. "-" denotes out of memory. See Table 16 for full results.

| Models | MiTSformer (Ours) | | iTrans. (2024) | | M-TCN. (2024) | | TimesNet (2023) | | PatchTST (2023) | | Cross. (2023) | | MICN (2023) | | LightTS (2023) | | Dlinear (2023) | | FiLM (2022) | | FED. (2022) | | Pyra. (2022) | |
|---|---|---|---|---|---|---|---|---|---|---|---|---|---|---|---|---|---|---|---|---|---|---|---|---|
| Metric | MAE | MSE | MAE | MSE | MAE | MSE | MAE | MSE | MAE | MSE | MAE | MSE | MAE | MSE | MAE | MSE | MAE | MSE | MAE | MSE | MAE | MSE | MAE | MSE |
| ETTm1 | **0.376** | **0.328** | 0.385 | 0.340 | 0.380 | 0.334 | 0.403 | 0.357 | 0.379 | 0.330 | 0.407 | 0.358 | 0.385 | 0.331 | 0.394 | 0.348 | 0.381 | 0.331 | 0.397 | 0.357 | 0.425 | 0.371 | 0.484 | 0.470 |
| ETTm2 | **0.365** | **0.363** | 0.371 | 0.373 | 0.366 | 0.371 | 0.376 | 0.395 | 0.368 | 0.364 | 0.878 | 1.056 | 0.528 | 0.599 | 0.529 | 0.578 | 0.512 | 0.568 | 0.376 | 0.389 | 0.386 | 0.385 | 0.896 | 1.732 |
| ETTh1 | **0.414** | **0.373** | 0.427 | 0.393 | 0.415 | 0.388 | 0.446 | 0.412 | 0.416 | 0.381 | 0.425 | 0.376 | 0.448 | 0.404 | 0.478 | 0.465 | 0.417 | 0.376 | 0.443 | 0.430 | 0.437 | 0.390 | 0.526 | 0.543 |
| ETTh2 | **0.430** | **0.449** | 0.442 | 0.472 | 0.442 | 0.480 | 0.453 | 0.490 | 0.440 | 0.464 | 0.917 | 1.422 | 0.692 | 1.000 | 0.729 | 1.063 | 0.675 | 0.982 | 0.450 | 0.490 | 0.476 | 0.508 | 1.304 | 2.548 |
| Weather | **0.326** | 0.268 | 0.334 | 0.279 | 0.328 | 0.269 | 0.348 | 0.296 | 0.332 | 0.276 | 0.338 | **0.257** | 0.356 | 0.278 | 0.354 | 0.277 | 0.346 | 0.274 | 0.353 | 0.291 | 0.386 | 0.322 | 0.357 | 0.274 |
| Exchange | 0.445 | 0.398 | 0.452 | 0.412 | 0.448 | 0.403 | 0.503 | 0.498 | 0.453 | 0.417 | 0.596 | 0.632 | 0.412 | 0.323 | 0.459 | 0.402 | **0.409** | **0.318** | 0.449 | 0.398 | 0.564 | 0.599 | 0.650 | 0.679 |
| ILI | **0.779** | **1.482** | 0.995 | 2.132 | 0.922 | 1.957 | 0.891 | 2.015 | 0.973 | 2.140 | 1.140 | 2.962 | 1.358 | 2.360 | 1.734 | 5.432 | 1.340 | 3.197 | 1.188 | 2.702 | 1.267 | 3.003 | 1.096 | 2.747 |
| Electric. | **0.260** | **0.168** | 0.293 | 0.207 | 0.270 | 0.174 | 0.288 | 0.187 | 0.295 | 0.207 | 0.326 | 0.237 | 0.295 | 0.185 | 0.339 | 0.240 | 0.314 | 0.220 | 0.316 | 0.234 | 0.351 | 0.248 | 0.400 | 0.319 |
| Traffic | **0.312** | **0.499** | 0.372 | 0.593 | 0.366 | 0.635 | 0.354 | 0.803 | 0.360 | 0.603 | - | - | 0.355 | 0.692 | 0.465 | 0.824 | 0.421 | 0.742 | - | - | 0.416 | 0.774 | 0.456 | 0.945 |

Due to the missing values, the imputation task requires the model to deeply exploit the underlying temporal dependencies and spatial correlations from partial observations. According to Table 2, MiTSformer achieves the best performance on most tasks. Through the recovery of LCVs and the intra- and inter-variable modality attention mechanisms, MiTSformer can effectively bridge spatial-temporal information interactions between CVs and DVs. In this way, not only can CVs exploit useful information for self-imputation by mining temporal and spatial correlation themselves, but also DVs can provide reliable auxiliary information for more accurate imputation of CVs.

**(5) Mixed Time Series Long-term Forecasting** We follow the settings of prediction lengths and benchmarks as [40], including ETT [49], Electricity (UCI), Weather (Wetterstation), Exchange [23] and ILI (CDC), corresponding to different applications. We focus on forecasting both DVs and CVs.

Since CVs contain more fine-grained information granularity and can more adequately evaluate the model forecasting performance, we mainly focus on the prediction accuracy of CVs, which are summarized in Table. 3. As reported, MiTSformer presents the best performance on most tasks (76 % according to Table 16), surpassing extensive advanced MLP-based, Transformer-based and CNN-based models. In addition, recent baselines- modernTCN and iTransformer present great performance due to their delicate design of global temporal receptive fields.

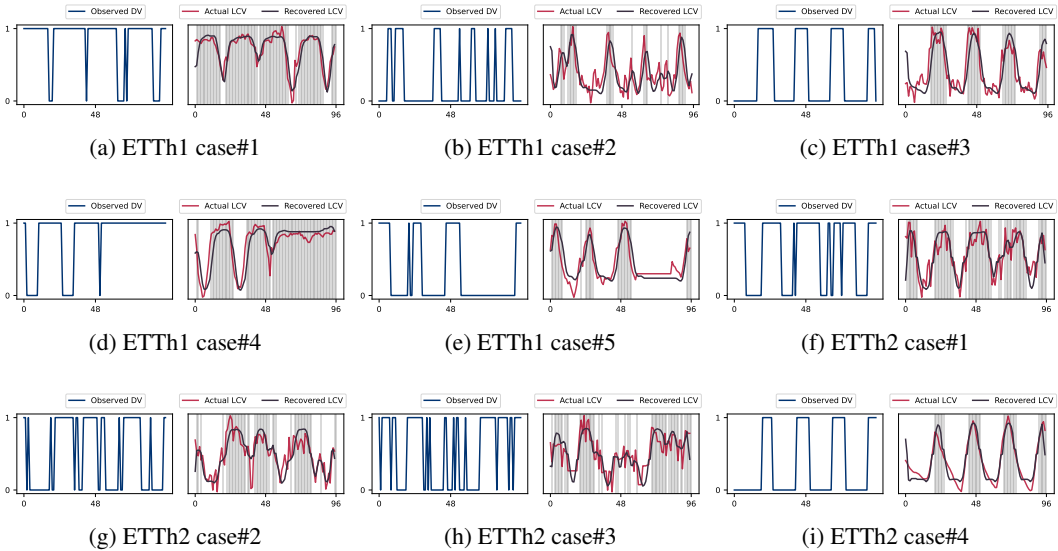

(a) ETTh1 case#1      (b) ETTh1 case#2      (c) ETTh1 case#3

(d) ETTh1 case#4      (e) ETTh1 case#5      (f) ETTh2 case#1

(g) ETTh2 case#2      (h) ETTh2 case#3      (i) ETTh2 case#4

Figure 9: Visualization of LCV recovery. For each subfigure, the *Left* plots the observed DVs, and the *Right* plots the actual LCVs (red line) and recovered LCVs (black line). The grey rectangular patches denotes the area where the observed DV is "1".

Table 4: Ablation analysis. For anomaly detection tasks, we do not ablate $\mathcal{L}_{\text{Rec}}$, as it is needed to support anomaly criterion calculation. The corresponding results are omitted with " / ".

| Models | Components | | | | Cls. (Acc.) | Long-term-forecasting 96-720 (MAE) | | | | | Anomaly detection (F1) | | | |
|---|---|---|---|---|---|---|---|---|---|---|---|---|---|---|
| | $\mathcal{L}_{\text{Dis}}$ | $\mathcal{L}_{\text{Smooth}}$ | $\mathcal{L}_{\text{Rec}}$ | Cross-Att | UCR Avg. | ETTm1 | ETTm2 | ETTh1 | ETTh2 | Avg. | SMD | MSL | SMAP | Avg. |
| MiTSformer | ✓ | ✓ | ✓ | ✓ | **71.9** | 0.415 | 0.439 | 0.442 | 0.455 | **0.438** | 87.84 | 85.30 | 96.83 | **89.99** |
| w/o $\mathcal{L}_{\text{Dis}}$ | ✗ | ✓ | ✓ | ✓ | 70.1 | 0.416 | 0.440 | 0.446 | 0.459 | 0.440 | 86.56 | 83.24 | 95.70 | 88.50 |
| w/o $\mathcal{L}_{\text{Smooth}}$ | ✓ | ✗ | ✓ | ✓ | 70.7 | 0.416 | 0.437 | 0.445 | 0.463 | 0.440 | 87.58 | 82.41 | 95.73 | 88.57 |
| w/o $\mathcal{L}_{\text{Rec}}$ | ✓ | ✓ | ✗ | ✓ | 69.7 | 0.463 | 0.478 | 0.624 | 0.509 | 0.519 | / | / | / | / |
| w/o LCVs | ✗ | ✗ | ✗ | ✓ | 69.3 | 0.501 | 0.479 | 0.626 | 0.509 | 0.529 | / | / | / | / |
| w/o Cross-Att | ✓ | ✓ | ✓ | ✗ | 70.2 | 0.428 | 0.442 | 0.451 | 0.461 | 0.446 | 84.92 | 84.10 | 95.79 | 88.27 |

## 4.2 Ablation Studies

To further verify the effectiveness of each key design, we conduct ablation experiments on mixed time series classification, long-term forecasting, and anomaly detection tasks. The results are summarized in Table 4, where "w/o LCVs" denotes removing $\mathcal{L}_{\text{Dis}}$, $\mathcal{L}_{\text{Smooth}}$, and $\mathcal{L}_{\text{Rec}}$ together, and "w/o Cross-Att" denotes removing the cross-attention sub-block and adopt self-attention sub-block solely.

**Latent Continuity Recovery** The three loss terms, $\mathcal{L}_{\text{Dis}}$, $\mathcal{L}_{\text{Smooth}}$, and $\mathcal{L}_{\text{Rec}}$ support the latent continuity recovery from different perspectives and work collaboratively. As shown in Table 4, ablating each loss term would lead to performance degradation for different tasks. Remarkably, the employment of all recovery loss functions in conjunction yields the optimal result.

**Attention Backbones** The cross-attention block is devised to bridge the information prorogation among LCVs and CVs by modeling spatial-temporal dependencies across LCVs and CVs, as LCVs and CVs provide complementary information for analysis tasks. The results in Table 4 also emphasize the importance of cross-variable-modality dependency modeling for MiTS analysis.

## 4.3 Visualization and Model Investigations

**Visualization of the Recovered LCVs** To verify the interpretability of LCV recovery, we provide visualization plots in Fig. 9. We find that DVs commonly show patterns of "*sudden changes*" or "*steady states*", which are less informative for analyzing spatial-temporal correlations. Fortunately, MiTSformer not only recovers their latent fine-grained and informative temporal variation patterns but also further leverages them for spatial-temporal representation learning, thereby mitigating the spatial-temporal heterogeneity challenge and achieving superior performance.

**Additional Analysis** We analyze the model efficiency in the Appendix. B, showing MiTSformer maintains great performance and efficiency compared with most baselines. In addition, we investigate hyperparameter sensitivity in the Appendix C. The results demonstrate that (1) MiTSformer is relatively stable in the selection of odel capacity-related hyper-parameters $d_{model}$ and number of layers $L$; (2) MiTSformer is quite robust to the weights of loss items (i.e., $\lambda_1$, $\lambda_1$, and, $\lambda_3$), and moderate weights bring optimal performance.

## 5 Conclusion and Future Work

This paper focuses on a challenging yet seldomly explored problem in the time series community and provides a systematic and universal solution for mixed time series analysis. To address the spatial-temporal heterogeneity problem, we first reveal the LCVs behind DVs and try to recover them for sufficient and balanced spatial-temporal modeling. Accordingly, MiTSformer is developed as a task-general mixed time series analysis framework by leveraging the temporal similarities and spatial interactions between LCVs and CVs. MiTSformer can perform adaptive LCV recovery for DVs via the adversarial guidance of CVs as well as smooth constraints, and model complete spatial-temporal dependencies via self- and cross-attention blocks. With extensive empirical evaluations, MiTSformer shows great practicality, superiority, and versatility in five mainstream mixed time series analysis tasks. In the future, it is of interest to empower MiTSformer with advanced pre-training techniques and powerful large language models for broader applications of mixed time series.

## Acknowledgments

This work was supported by the National Natural Science Foundation of China (No. 62450020 and No. 62125306), and Zhejiang Key Research and Development Project (No. 2024C01163).

## Footnotes

[2]https://pytorch.org/

[3]https://github.com/thuml/Time-Series-Library

## References

[1] Cdc. illness. https://gis.cdc.gov/grasp/fluview/fluportaldashboard. html.

[2] Uci. electricity. https://archive.ics.uci.edu/ml/datasets/ electricityloaddiagrams20112014/.

[3] Wetterstation. weather. https://www.bgc-jena.mpg.de/wetter/.

[4] Ahmed Abdulaal, Zhuanghua Liu, and Tomer Lancewicki. Practical approach to asynchronous multivariate time series anomaly detection and localization. In *Proceedings of the 27th ACM SIGKDD conference on knowledge discovery & data mining*, pages 2485–2494, 2021.

[5] Anthony Bagnall, Hoang Anh Dau, Jason Lines, Michael Flynn, James Large, Aaron Bostrom, Paul Southam, and Eamonn Keogh. The uea multivariate time series classification archive, 2018. *arXiv preprint arXiv:1811.00075*, 2018.

[6] Shaojie Bai, J Zico Kolter, and Vladlen Koltun. An empirical evaluation of generic convolutional and recurrent networks for sequence modeling. *arXiv preprint arXiv:1803.01271*, 2018.

[7] Kaifeng Bi, Lingxi Xie, Hengheng Zhang, Xin Chen, Xiaotao Gu, and Qi Tian. Accurate medium-range global weather forecasting with 3d neural networks. *Nature*, 619(7970):533–538, 2023.

[8] David Campos, Miao Zhang, Bin Yang, Tung Kieu, Chenjuan Guo, and Christian S Jensen. Lightts: Lightweight time series classification with adaptive ensemble distillation. *Proceedings of the ACM on Management of Data*, 1(2):1–27, 2023.

[9] Alberto Castellini, Francesco Masillo, Davide Azzalini, Francesco Amigoni, and Alessandro Farinelli. Adversarial data augmentation for hmm-based anomaly detection. *IEEE Transactions on Pattern Analysis and Machine Intelligence*, 2023.

[10] Jiawei Chen, Pengyu Song, and Chunhui Zhao. Multi-scale self-supervised representation learning with temporal alignment for multi-rate time series modeling. *Pattern Recognition*, 145:109943, 2024.

[11] Jiawei Chen, Pengyu Song, Chunhui Zhao, and Jinliang Ding. Spatiotemporal multiscale correlation embedding with process variable reorder for industrial soft sensing. *IEEE Transactions on Instrumentation and Measurement*, 2023.

[12] Junhao Chen, Chunhui Zhao, and Jinliang Ding. A flexible probabilistic framework with concurrent analysis of continuous and categorical data for industrial fault detection and diagnosis. *IEEE Transactions on Industrial Informatics*, 2023.

[13] Kochenderfer M J. Chen Y C, Wheeler T A. Learning discrete bayesian networks from continuous data. In *Journal of Artificial Intelligence Research*, pages 59: 103–132, 2017.

[14] Kien Do, Truyen Tran, and Svetha Venkatesh. Multilevel anomaly detection for mixed data. *arXiv preprint arXiv:1610.06249*, 2016.

[15] Kien Do, Truyen Tran, and Svetha Venkatesh. Energy-based anomaly detection for mixed data. *Knowledge and Information Systems*, 57(2):413–435, 2018.

[16] Luo donghao and wang xue. ModernTCN: A modern pure convolution structure for general time series analysis. In *The Twelfth International Conference on Learning Representations*, 2024.

[17] Emadeldeen Eldele, Mohamed Ragab, Zhenghua Chen, Min Wu, Chee-Keong Kwoh, Xiaoli Li, and Cuntai Guan. Self-supervised contrastive representation learning for semi-supervised time-series classification. *IEEE Transactions on Pattern Analysis and Machine Intelligence*, 2023.

[18] Liangjun Feng and Chunhui Zhao. Fault description based attribute transfer for zero-sample industrial fault diagnosis. *IEEE Transactions on Industrial Informatics*, 17(3):1852–1862, 2021.

[19] Yaroslav Ganin and Victor Lempitsky. Unsupervised domain adaptation by backpropagation. In *International conference on machine learning*, pages 1180–1189. PMLR, 2015.

[20] Rong Hu, Ling Chen, Shenghuan Miao, and Xing Tang. Swl-adapt: An unsupervised domain adaptation model with sample weight learning for cross-user wearable human activity recognition. *Proceedings of the AAAI Conference on Artificial Intelligence*, 2023.

[21] Kyle Hundman, Valentino Constantinou, Christopher Laporte, Ian Colwell, and Tom Soderstrom. Detecting spacecraft anomalies using lstms and nonparametric dynamic thresholding. In *Proceedings of the 24th ACM SIGKDD international conference on knowledge discovery & data mining*, pages 387–395, 2018.

[22] Yuxin Jia, Youfang Lin, Xinyan Hao, Yan Lin, Shengnan Guo, and Huaiyu Wan. Witran: Water-wave information transmission and recurrent acceleration network for long-range time series forecasting. *Advances in Neural Information Processing Systems*, 36, 2024.

[23] Guokun Lai, Wei-Cheng Chang, Yiming Yang, and Hanxiao Liu. Modeling long-and short-term temporal patterns with deep neural networks. In *The 41st international ACM SIGIR conference on research & development in information retrieval*, pages 95–104, 2018.

[24] Qiude Li, Qingyu Xiong, Shengfen Ji, Yang Yu, Chao Wu, and Hualing Yi. A method for mixed data classification base on rbf-elm network. *Neurocomputing*, 431:7–22, 2021.

[25] Shengsheng Lin, Weiwei Lin, Wentai Wu, Feiyu Zhao, Ruichao Mo, and Haotong Zhang. Segrnn: Segment recurrent neural network for long-term time series forecasting. *arXiv preprint arXiv:2308.11200*, 2023.

[26] Shizhan Liu, Hang Yu, Cong Liao, Jianguo Li, Weiyao Lin, Alex X Liu, and Schahram Dustdar. Pyraformer: Low-complexity pyramidal attention for long-range time series modeling and forecasting. In *International conference on learning representations*, 2022.

[27] Yong Liu, Tengge Hu, Haoran Zhang, Haixu Wu, Shiyu Wang, Lintao Ma, and Mingsheng Long. itransformer: Inverted transformers are effective for time series forecasting. In *The Twelfth International Conference on Learning Representations*, 2024.

[28] Yong Liu, Haixu Wu, Jianmin Wang, and Mingsheng Long. Non-stationary transformers: Exploring the stationarity in time series forecasting. *Advances in Neural Information Processing Systems*, 35:9881–9893, 2022.

[29] Aditya P Mathur and Nils Ole Tippenhauer. Swat: A water treatment testbed for research and training on ics security. In *2016 international workshop on cyber-physical systems for smart water networks (CySWater)*, pages 31–36. IEEE, 2016.

[30] Zelin Ni, Hang Yu, Shizhan Liu, Jianguo Li, and Weiyao Lin. Basisformer: Attention-based time series forecasting with learnable and interpretable basis. *Advances in Neural Information Processing Systems*, 36, 2024.

[31] Yuqi Nie, Nam H Nguyen, Phanwadee Sinthong, and Jayant Kalagnanam. A time series is worth 64 words: Long-term forecasting with transformers. In *The Eleventh International Conference on Learning Representations*, 2023.

[32] Alec Radford, Jeffrey Wu, Rewon Child, David Luan, Dario Amodei, and Ilya Sutskever. Language models are unsupervised multitask learners. *OpenAI Blog*, 2019.

[33] Ya Su, Youjian Zhao, Chenhao Niu, Rong Liu, Wei Sun, and Dan Pei. Robust anomaly detection for multivariate time series through stochastic recurrent neural network. In *Proceedings of the 25th ACM SIGKDD international conference on knowledge discovery & data mining*, pages 2828–2837, 2019.

[34] Chang Wei Tan, Christoph Bergmeir, François Petitjean, and Geoffrey I Webb. Time series extrinsic regression: Predicting numeric values from time series data. *Data Mining and Knowledge Discovery*, 35(3):1032–1060, 2021.

[35] Giulio Ventura and Elena Benvenuti. *Advances in Discretization Methods: Discontinuities, Virtual Elements, Fictitious Domain Methods*, volume 12. Springer, 2016.

[36] Huiqiang Wang, Jian Peng, Feihu Huang, Jince Wang, Junhui Chen, and Yifei Xiao. Micn: Multi-scale local and global context modeling for long-term series forecasting. In *The Eleventh International Conference on Learning Representations*, 2023.

[37] Min Wang, Donghua Zhou, and Maoyin Chen. Anomaly monitoring of nonstationary processes with continuous and two-valued variables. *IEEE Transactions on Systems, Man, and Cybernetics: Systems*, 53(1):49–58, 2022.

[38] Min Wang, Donghua Zhou, and Maoyin Chen. Hybrid variable monitoring: An unsupervised process monitoring framework with binary and continuous variables. *Automatica*, 147:110670, 2023.

[39] Shiyu Wang, Haixu Wu, Xiaoming Shi, Tengge Hu, Huakun Luo, Lintao Ma, James Y Zhang, and JUN ZHOU. Timemixer: Decomposable multiscale mixing for time series forecasting. In *The Twelfth International Conference on Learning Representations*, 2024.

[40] Haixu Wu, Tengge Hu, Yong Liu, Hang Zhou, Jianmin Wang, and Mingsheng Long. Timesnet: Temporal 2d-variation modeling for general time series analysis. In *The Eleventh International Conference on Learning Representations*, 2023.

[41] Haixu Wu, Jiehui Xu, Jianmin Wang, and Mingsheng Long. Autoformer: Decomposition transformers with auto-correlation for long-term series forecasting. *Advances in neural information processing systems*, 34:22419–22430, 2021.

[42] Haixu Wu, Hang Zhou, Mingsheng Long, and Jianmin Wang. Interpretable weather forecasting for worldwide stations with a unified deep model. *Nature Machine Intelligence*, 5(6):602–611, 2023.

[43] Wanke Yu, Chunhui Zhao, and Biao Huang. Moninet with concurrent analytics of temporal and spatial information for fault detection in industrial processes. *IEEE Transactions on Cybernetics*, 52(8):8340–8351, 2022.

[44] Jiaqi Yue, Jiancheng Zhao, and Chunhui Zhao. Similarity makes difference: Sshtn for generalized zero-shot industrial fault diagnosis by leveraging auxiliary set. *IEEE Transactions on Industrial Informatics*, 2024.

[45] Ailing Zeng, Muxi Chen, Lei Zhang, and Qiang Xu. Are transformers effective for time series forecasting? In *Proceedings of the AAAI conference on artificial intelligence*, volume 37, pages 11121–11128, 2023.

[46] Yunhao Zhang and Junchi Yan. Crossformer: Transformer utilizing cross-dimension dependency for multivariate time series forecasting. In *The Eleventh International Conference on Learning Representations*, 2023.

[47] Chunhui Zhao. Perspectives on nonstationary process monitoring in the era of industrial artificial intelligence. *Journal of Process Control*, 116:255–272, 2022.

[48] Chunhui Zhao, Junhao Chen, and Hua Jing. Condition-driven data analytics and monitoring for wide-range nonstationary and transient continuous processes. *IEEE Transactions on Automation Science and Engineering*, 18(4):1563–1574, 2021.

[49] Haoyi Zhou, Shanghang Zhang, Jieqi Peng, Shuai Zhang, Jianxin Li, Hui Xiong, and Wancai Zhang. Informer: Beyond efficient transformer for long sequence time-series forecasting. In *Proceedings of the AAAI conference on artificial intelligence*, volume 35, pages 11106–11115, 2021.

[50] Tian Zhou, Ziqing Ma, Qingsong Wen, Liang Sun, Tao Yao, Wotao Yin, Rong Jin, et al. Film: Frequency improved legendre memory model for long-term time series forecasting. *Advances in Neural Information Processing Systems*, 35:12677–12690, 2022.

[51] Tian Zhou, Ziqing Ma, Qingsong Wen, Xue Wang, Liang Sun, and Rong Jin. Fedformer: Frequency enhanced decomposed transformer for long-term series forecasting. In *International Conference on Machine Learning*, pages 27268–27286. PMLR, 2022.

[52] Tian Zhou, Peisong Niu, Liang Sun, Rong Jin, et al. One fits all: Power general time series analysis by pretrained lm. *Advances in neural information processing systems*, 36, 2024.

# A Implementation Details

## A.1 Datasets and Experimental Platforms

Despite being a fundamental issue, modeling mixed time series remains underexplored in academia, with a lack of specialized benchmark datasets for mixed time series. To meet the tasks for mixed time series, we employed benchmark time-series datasets and implemented discretization to convert some CVs into DVs. Our conversion process simulates the generation process of DVs and discretizes variables while preserving their inherent coupling relationships and properties. We summarized the dataset descriptions in Table 5. All experiments are repeated three times, implemented in PyTorch [2] and conducted on Linux servers with Intel(R) Xeon(R) Gold 6246 CPUs and NVIDIA 3090 24GB GPUs. The versions of Python and Pytorch are 3.9.7, and 1.10.0 respectively.

## A.2 Hyperparameters and Experimental Configurations

All the baselines that we reproduced are implemented based on their official codes or Time Series Library (TSlib)[3]. Since some baseline models are designed for specific analysis tasks, we modify them to serve different analysis tasks by replacing task heads and loss functions. Specifically, we keep the original backbone architecture for each method as the feature extractor, and we adopt universal task-specific heads and loss functions consistently for all methods. The architectures of task heads and related loss functions will be introduced in the following section.

The hyper-parameter configurations of MiTSformer are summarized in Table 6 and the hyper-parameter configurations of baseline models are summarized in Table 7. Some hyperparameters, including batch size, training epochs, dropout rate, and the number of attention heads (transformer-based models only) are fixed and kept the same for all baseline models and MiTSformer. Moreover, to compare the upper bound of different models specifically for mixed time series analysis tasks, we conduct grid searches of model capacity-related hyper-parameters, including number of layers and $d_{model}$ / hidden size and optimization-related hyper-parameter-learning rate. Compared with baseline models, MiTSformer additionally fine-tuned the weight of smoothness loss weight $\lambda_1$ to adapt to different temporal variations of different datasets. The variable modality discrimination loss weight is fixed at 1.0 and the reconstruction loss weight is fixed at 1.0 for all experiments.

## A.3 Algorithm Implementations of MiTSformer

Mixed time series modeling differs significantly from typical time series tasks due to variable heterogeneity. Thereby, each component in MiTSformer is essential to address it via latent continuity recovery and alignment. We summarized the model feed-forward, loss-calculation, and parameter-update procedures of MiTSformer with pseudo-codes, which are presented in Algorithm 1. We also report the standard deviation of MiTSformer performance with different random seeds in Table 8, which exhibits that the performance of MiTSformer is stable. Here we introduce the detailed architectures of each module in MiTSformer.

**Recovery Network** The recovery network, receiving the input of DVs $X^D$ with shape (Batchsize $\times n \times T$) and outputting LCVs $X^{LCV}$ with shape (Batchsize $\times n \times T$), is composed of 3 dilated convolution blocks with residual connections. Specifically, the $l$th dilated convolution block is composed of 2 dilated 1D convolutional layers with kernel_size $= 2$, hidden_channel $= 8$, dilation $= 2^l$ and GELU activations. Additionally, padding operation Padding $= ((\text{kernel\_size} - 1) \times \text{dilation} + 1) //2$ is adopted for the convolutional layer to ensure that the input and output have the same length $T$, where $(\text{kernel\_size} - 1) \times \text{dilation} + 1$ denotes the receptive field). Finally, we perform z-score normalization for each $x^{LC} \in \mathbb{R}^{1 \times T}$ to ensure training stability as

$$x^{LC} = \frac{x^{LC} - \text{Mean}(x^{LC})}{\text{Std}(x^{LC})}$$

where Mean and Std denote the mean and the standard deviation along the time axis, respectively.

**Self-attention and Cross-attention Sub-blocks** The self-attention sub-block is composed of vanilla FullAttention with 8 attention heads and 0.1 dropout rate. The same for the cross-attention sub-block.

Table 5: Dataset descriptions. The dataset size is organized in (Train, Validation, Test). "Dim. $p$" denotes the total variable dimension and "Dim. $n$" denotes the discrete variable dimension. Since current benchmark datasets are time series encompassing only continuous variables, we generate mixed time series from these datasets by discretizing partial variables. For each dataset, we randomly select half variables as DVs ($n = \lfloor 0.5p \rfloor$), whose values are first MinMax normalized and then discretized into the value of 0 or 1 with the threshold 0.5 as $\texttt{int}(\texttt{MinMax}(x) > 0.5)$.

| Task | Dataset | Dim $p$ | Dim $n$ | Series Length | Dataset Size | Semantics |
|------|---------|---------|---------|---------------|--------------|-----------|
| Classification (UEA) | EthanolConcentration | 3 | 1 | 1751 | (261, 0, 263) | Alcohol Industry |
| | FaceDetection | 144 | 72 | 62 | (5890, 0, 3524) | Face (250Hz) |
| | Handwriting | 3 | 1 | 152 | (150, 0, 850) | Handwriting |
| | Heartbeat | 61 | 30 | 405 | (204, 0, 205) | Heart Beat |
| | JapaneseVowels | 12 | 6 | 29 | (270, 0, 370) | Voice |
| | PEMS-SF | 963 | 481 | 144 | (267, 0, 173) | Transportation (Daily) |
| | SelfRegulationSCP1 | 6 | 3 | 896 | (268, 0, 293) | Health (256Hz) |
| | SelfRegulationSCP2 | 7 | 3 | 1152 | (200, 0, 180) | Health (256Hz) |
| | SpokenArabicDigits | 13 | 6 | 93 | (6599, 0, 2199) | Voice (11025Hz) |
| | UWaveGestureLibrary | 3 | 1 | 315 | (120, 0, 320) | Gesture |
| Extrinsic Regression (UCR) | AppliancesEnergy | 24 | 12 | 144 | (96,0,42) | Energy Monitoring |
| | HouseholdPower.1 | 5 | 2 | 1440 | (746,0,694) | Energy Monitoring |
| | HouseholdPower.2 | 5 | 2 | 1440 | (746,0,694) | Energy Monitoring |
| | BenzeneConcentration | 8 | 4 | 240 | (3433,0,5445) | Environment Monitoring |
| | BeijingPM25Quality | 9 | 4 | 24 | (12432,0,5100) | Environment Monitoring |
| | BeijingPM10Quality | 9 | 4 | 24 | (12432,0,5100) | Environment Monitoring |
| | LiveFuelMoisture. | 7 | 3 | 365 | (3493,0,1510 ) | Environment Monitoring |
| | AustraliaRainfall | 3 | 1 | 24 | (112186,0,48081) | Environment Monitoring |
| | PPGDalia | 4 | 2 | 256 | (43215,0,21482) | Health Monitoring |
| | IEEEPPG | 5 | 2 | 1000 | (1768,0,1328) | Health Monitoring |
| Imputation | ETTm1,ETTm2 | 7 | 3 | 96 | (34465, 11521, 11521) | Electricity (15 mins) |
| | ETTh1,ETTh2 | 7 | 6 | 96 | (8545, 2881, 2881) | Electricity (15 mins) |
| | Electricity | 321 | 160 | 96 | (18317, 2633, 5261) | Electricity (Hourly) |
| | Weather | 21 | 10 | 96 | (36792, 5271, 10540) | Weather (10 mins) |
| Anomaly Detection | SMD | 38 | 19 | 100 | (566724, 141681, 708420) | Server Machine |
| | MSL | 55 | 27 | 100 | (44653, 11664, 73729) | Spacecraft |
| | SMAP | 25 | 12 | 100 | (108146, 27037, 427617) | Spacecraft |
| | SWaT | 51 | 25 | 100 | (396000, 99000, 449919) | Infrastructure |
| | PSM | 25 | 12 | 100 | (105984, 26497, 87841) | Server Machine |
| Long-term Forecasting | ETTm1,ETTm2 | 7 | 3 | {96;96,192,336,720} | (34465, 11521, 11521) | Electricity (15 mins) |
| | ETTh1,ETTh2 | 7 | 6 | {96;96,192,336,720} | (8545, 2881, 2881) | Electricity (15 mins) |
| | Electricity | 321 | 160 | {96;96,192,336,720} | (18317, 2633, 5261) | Electricity (Hourly) |
| | Traffic | 862 | 431 | {96;96,192,336,720} | (12185, 1757, 3509) | Transportation (Hourly) |
| | Weather | 21 | 10 | {96;96,192,336,720} | (36792, 5271, 10540) | Weather (10 mins) |
| | Exchange | 8 | 4 | {96;96,192,336,720} | (5120, 665, 1422) | Exchange rate (Daily) |
| | ILI | 7 | 3 | {24;24,36,48,60} | (617, 74, 170) | Illness (Weekly) |

Table 6: Experiment configuration of MiTSformer. All the experiments use the ADAM optimizer with the default hyperparameter configuration for $(\beta_1, \beta_2)$ as (0.9, 0.999) with proper early stopping, and adopt a dropout rate of 0.1. $\lambda_1$ denotes the weight of smoothness loss, $\lambda_2$ denotes the weight of reconstruction loss, and $\lambda_3$ denotes the weight of variable modality discrimination loss. LR$^*$ denotes the initial learning rate. The number of attention heads is set to 8 for all experiments.

| Task | Model Hyperparameters | | | | | Training Process | | |
|---|---|---|---|---|---|---|---|---|
| | Layers | $d_{model}$ | $\lambda_1$ | $\lambda_2$ | $\lambda_3$ | LR$^*$ | Batch size | Epochs |
| Classification | {1,2} | {16,32,64,128,256} | {0.1,0.3,0.5} | 1 | 1 | $\{10^{-3}, 5*10^{-4}, 10^{-4}\}$ | 64/16$^\dagger$ | 100 |
| Extrinsic Regression | {1,2} | {16,32,64,128,256} | {0.1,0.3,0.5} | 1 | 1 | $\{10^{-3}, 5*10^{-4}, 10^{-4}\}$ | 64 | 100 |
| Imputation | {1,2} | {64,128,256} | {0.3,0.5} | 1 | 1 | $\{10^{-3}, 10^{-4}\}$ | 32 | 10 |
| Anomaly Detection | {1,2} | {64,128,256} | {0.3,0.5} | 1 | 1 | $\{10^{-3}, 10^{-4}\}$ | 32 | 10 |
| Long-term Forecasting | {1,2} | {64,128,256} | {0.3,0.5} | 1 | 1 | $\{10^{-3}, 10^{-4}\}$ | 32 | 10 |

$^\dagger$We set the batch size to 16 for the PEMS-SF dataset due to its high dimensionality. We set the batch size to 64 for other classification datasets. The same for Table .7

Table 7: Experiment configuration of baseline models. All the experiments use the ADAM optimizer with the default hyperparameter configuration for $(\beta_1, \beta_2)$ as (0.9, 0.999) with proper early stopping, and adopt a dropout rate of 0.1. LR$^*$ denotes the initial learning rate. For Transformer-based models, the number of attention heads is set to 8 for all experiments.

| Task | Model Hyperparameters | | Training Process | | |
|---|---|---|---|---|---|
| | Layers | $d_{model}$ / hidden size | LR$^*$ | Batch size | Epochs |
| Classification | {1,2} | {16,32,64,128,256} | $\{10^{-3}, 5*10^{-4}, 10^{-4}\}$ | 64/16$^\dagger$ | 100 |
| Extrinsic Regression | {1,2} | {16,32,64,128,256} | $\{10^{-3}, 5*10^{-4}, 10^{-4}\}$ | 64 | 100 |
| Imputation | {1,2} | {64,128,256} | $\{10^{-3}, 10^{-4}\}$ | 32 | 10 |
| Anomaly Detection | {1,2} | {64,128,256} | $\{10^{-3}, 10^{-4}\}$ | 32 | 10 |
| Long-term Forecasting | {1,2} | {64,128,256} | $\{10^{-3}, 10^{-4}\}$ | 32 | 10 |

**Variable Modality Discriminator** The variable modality discriminator, receiving the input variate embeddings with shape (Batchsize $\times p \times d_{model}$) and outputting the variable modality class with shape (Batchsize $\times p \times 2$), is composed of 3-layer MLP with structure of {d_model, 4 * d_model, 4 * d_model, 2}. Relu activation and batch normalization are also adopted.

**Reconstruction Decoders** The discrete variable reconstruction decoder, receiving the input of LCV embeddings (Batchsize $\times n \times d_{model}$) and outputting the reconstructed DVs (Batchsize $\times n \times T \times 2$), is composed of a linear layer with input_size = d_model, output_size = $2*T$ (the discrete variable reconstruction can be treated as a binary classification task). The continuous variable reconstruction decoder, receiving the input of CV embeddings (Batchsize $\times (p-n) \times d_{model}$) and outputting the reconstructed CVs (Batchsize $\times (p-n) \times T$), is composed of a linear layer with input_size = d_model, output_size = $T$.

In the following, we will introduce the architectures of task heads and loss functions for different downstream tasks.

## A.4  Pipeline for Classification

Classification is a long-standing task in the time series community and is widely used to evaluate the high-level representation capacity of models. The overall pipeline of MiTSformer-based classification is depicted in Fig. 10. For the classification task, both token embeddings of LCVs and CVs provide complementary information. Thereby, we concatenate the embeddings of LCVs and CVs. The fused embeddings are flattened and fed into a classifier network to predict the class labels. For MiTSformer and baseline models, the classifier is composed of a single-layer MLP with GELU activation and

**Algorithm 1** The training process of MiTSformer

**Input:** Input mixed time series with continuous variables $X^C = \{x_1, x_2, ..., x_{p-n}\} \in \mathbb{R}^{(p-n)\times T}$, and discrete variables $X^D = \{x_1, x_2, ..., x_n\} \in^{n \times T}$, number of attention blocks $L$, loss weights $\lambda_1$, $\lambda_2$ and $\lambda_3$.

**Output:** Optimized model parameters of MiTSformer.

1: Initialize the parameters of MiTSformer;
2: **while** not converge **do**
3:    *(1) Feedforward Computation:*
4:    ▷ Latent Continuity Recovery for DVs
5:    $x^{LC} \leftarrow \text{Rec-Net}(x^D)$;
6:    ▷ Variate-wise Token Embedding of LCVs and CVs
7:    $z_0^{LC} \leftarrow \text{Embed}_{\text{LCV}}(x^{LC})$
8:    $z_0^C \leftarrow \text{Embed}_{\text{CV}}(x^C)$
9:    ▷ Inter- and Intra-Variable-Modality Spatial-Temporal Modeling
10:    **for** $l = [0, 1, ..., L-1]$ **do**
11:      ▷ Intra-Variable-Modality Self-Attention Sub-Block
12:      $\hat{z}_l^C \leftarrow \text{LN}\left(z_l^C + \text{Self-Attn}\left(\left[Q_l^C, K_l^C, V_l^C\right]\right)\right)$
13:      $\hat{z}_l^C \leftarrow \text{LN}\left(\hat{z}_l^C + \text{FFN}\left(\hat{z}_l^C\right)\right)$
14:      $\hat{z}_l^{LC} \leftarrow \text{LN}\left(z_l^{LC} + \text{Self-Attn}\left(\left[Q_l^{LC}, K_l^{LC}, V_l^{LC}\right]\right)\right)$
15:      $\hat{z}_l^{LC} \leftarrow \text{LN}\left(\hat{z}_l^{LC} + \text{FFN}\left(\hat{z}_l^{LC}\right)\right)$
16:      ▷ Inter-Variable-Modality Cross-Attention Sub-Block
17:      $z_{l+1}^C \leftarrow \text{LN}\left(\hat{z}_l^C + \text{Cross-Attn}\left(\left[Q_l^C, K_l^{LC}, V_l^{LC}\right]\right)\right)$
18:      $z_{l+1}^C \leftarrow \text{LN}\left(z_{l+1}^C + \text{FFN}\left(z_{l+1}^C\right)\right)$
19:      $z_{l+1}^{LC} \leftarrow \text{LN}\left(z_{l+1}^{LC} + \text{Cross-Attn}\left(\left[Q_l^{LC}, K_l^C, V_l^C\right]\right)\right)$
20:      $z_{l+1}^{LC} \leftarrow \text{LN}\left(z_{l+1}^{LC} + \text{FFN}\left(z_{l+1}^{LC}\right)\right)$
21:    **end for**
22:    ▷ Task Prediction (if needed)
23:    $\hat{y} \leftarrow \text{Task-Heads}(z_L^{LC}, z_L^C)$
24:    *(2) Loss Calculation:*
25:    ▷ Task Supervision Loss (if needed)
26:    $\mathcal{L}_{\text{Task}} \leftarrow \text{Task-Criterion}(\hat{y}, y)$.
27:    ▷ Smoothness Constraint Loss
28:    $\mathcal{L}_{\text{smooth}} = \left\|\text{Abs}\left(Sx^D\right) \otimes \left(Sx^{LC}\right)\right\|_2^2$.
29:    ▷ Reconstruction Loss
30:    $\mathcal{L}_{\text{Rec}} = \sum_{i=1}^{p-n} \text{MSE}(\text{Rec-Decoder}\left(z_{L,i}^C\right), x_i^C) + \sum_{i=1}^{n} \text{CE}(\text{Rec-Decoder}\left(z_{L,i}^{LC}\right), x_i^D)$.
31:    ▷ Variable Modality Discrimination Loss (via Gradient Reverse Layer)
32:    $\mathcal{L}_{\text{Dis}} = \mathbb{E}\left[\log\left(\text{GRL}(\text{Dis}\left(z^C\right))\right)\right] + \mathbb{E}\left[\log\left(1 - \text{GRL}(\text{Dis}\left(z^{LC}\right))\right)\right]$
33:    ▷ Overall Loss
34:    $\mathcal{L}_{\text{All}} = \mathcal{L}_{\text{Task}} + \lambda_1\mathcal{L}_{\text{Smooth}} + \lambda_2\mathcal{L}_{\text{Rec}} + \lambda_3\mathcal{L}_{\text{Dis}}$
35:    *(3) Parameter Update:*
36:    Update the parameters of models using Adam optimizer to minimize $\mathcal{L}_{all}$;
37: **end while**
38: **return** Optimized model parameters.

Table 8: Robustness of MiTSformer performance on forecasting datasets. Averaged MAE, MSE, and their standard deviations based on different random seeds are reported.

| Dataset | ETTh1 | | ETTm1 | | ETTm2 | |
|---|---|---|---|---|---|---|
| Horizon | MAE | MSE | MAE | MSE | MAE | MSE |
| 96 | 0.381±0.001 | 0.323±0.000 | 0.342±0.001 | 0.271±0.000 | 0.293±0.002 | 0.231±0.003 |
| 192 | 0.408±0.000 | 0.369±0.000 | 0.363±0.003 | 0.31±0.002 | 0.344±0.001 | 0.324±0.002 |
| 336 | 0.424±0.002 | 0.402±0.002 | 0.382±0.002 | 0.338±0.003 | 0.383±0.002 | 0.395±0.001 |
| 720 | 0.442±0.004 | 0.398±0.002 | 0.415±0.003 | 0.391±0.003 | 0.439±0.003 | 0.503±0.002 |
| Dataset | Electricity | | Weather | | Traffic | |
| Horizon | MAE | MSE | MAE | MSE | MAE | MSE |
| 96 | 0.235±0.002 | 0.143±0.001 | 0.231±0.000 | 0.147±0.001 | 0.292±0.002 | 0.459±0.002 |
| 192 | 0.250±0.000 | 0.159±0.002 | 0.294±0.000 | 0.216±0.001 | 0.310±0.003 | 0.494±0.002 |
| 336 | 0.265±0.002 | 0.171±0.001 | 0.353±0.001 | 0.298±0.001 | 0.316±0.003 | 0.508±0.003 |
| 720 | 0.291±0.003 | 0.198±0.002 | 0.426±0.002 | 0.412±0.001 | 0.330±0.005 | 0.534±0.003 |

a dropout layer with a dropout rate of 0.1. The task supervision loss is Cross-Entropy Loss. The classification accuracy is used as the performance evaluation metric.

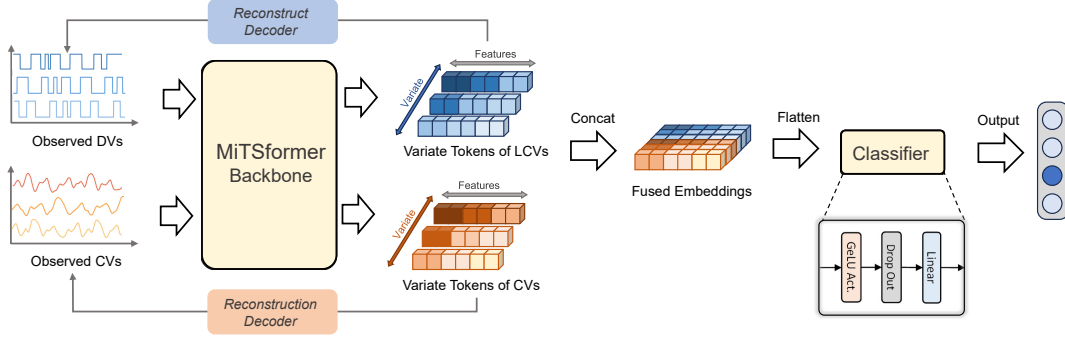

Figure 10: Overall pipeline of MiTSformer-based classification. The embeddings of LCVs and CVs are concatenated, flattened, and fed into the classifier for classification.

## A.5 Pipeline for Extrinsic Regression

Closely related to classification, time series extrinsic regression aims to learn the relationship between a time series and a continuous scalar variable. The overall pipeline of MiTSformer-based extrinsic regression is depicted in Fig. 11. Similar to classification, both token embeddings of LCVs and CVs provide complementary information for regression tasks. Thereby, we concatenate the embeddings of LCVs and CVs. The fused embeddings are flattened and fed into a regressor network to predict the numerical values. For MiTSformer and baseline models, the regressor is composed of a single-layer MLP with GELU activation and a dropout layer with a dropout rate of 0.1. The task supervision loss is MSE Loss. We adopt the mean absolute error (RMAE) and the root mean square error (RMSE) as the performance evaluation metrics as

$$\text{RMSE} = \sqrt{\frac{1}{N_y} \sum_{i=1}^{N_y} (\hat{y}_i - y_i)^2} \tag{7}$$

$$\text{MAE} = \frac{1}{N_y} \sum_{i=1}^{N_y} |\hat{y}_i - y_i| \tag{8}$$

where $N_y$ denotes the number of test samples, $\hat{y}_i$ denotes the predicted labels and $y_i$ denotes the ground-truth label.

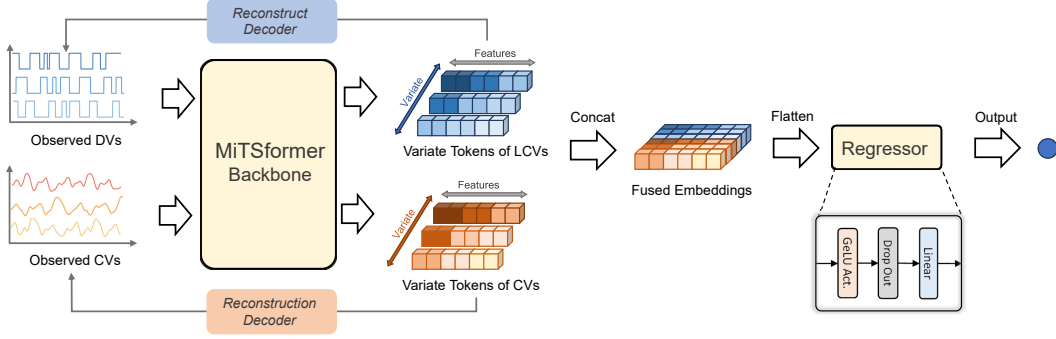

Figure 11: Overall pipeline of MiTSformer-based extrinsic regression. The embeddings of LCVs and CVs are concatenated, flattened, and fed into the regressor for regression.

## A.6 Pipeline for Imputation

The imputation techniques are developed to impute the missing values based on the partially observed time points. For the imputation task, we mainly focus on imputing the missing value of CVs, while DVs are adopted as input to provide auxiliary information. The overall pipeline of MiTSformer-based imputation is depicted in Fig. 11. For MiTSformer, the task head is an imputation decoder composed of a linear layer, which performs variate-wise imputation for each CV. For baseline methods that adopt the channel-independent strategy, we adopt the same task head as MiTSformer. For baseline methods that adopt the channel-mixing strategy, we adopt a linear layer with input size of $d_{model}$ and output size of continuous variable dimension $p - n$ to reconstruct the CVs. We use a mask matrix $M \in \mathbb{R}^{T \times (p-n)}$ to represent the missing values in input CVs. The state of $M$ denotes whether the corresponding element value of CVs is missing (denoted by 0) or not (denoted by 1). The task supervision loss is the MSE loss calculated on the masked observations. Specifically for MiTSformer, the reconstruction loss of CVs is calculated on unmasked observations. We adopt the mean absolute error (MAE) and the root mean square error (RMSE) as the performance evaluation metrics that are computed on masked elements.

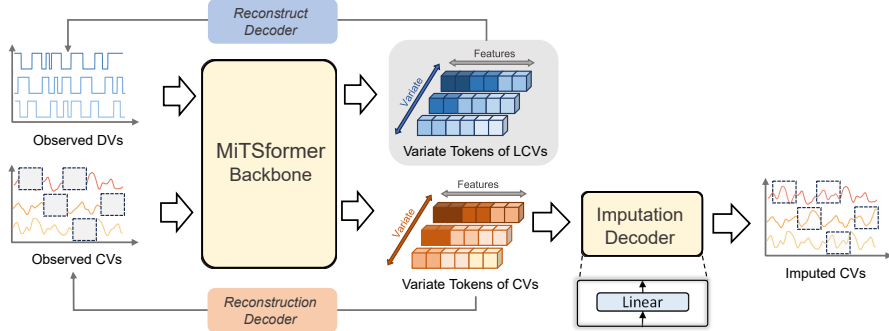

Figure 12: Overall pipeline of MiTSformer-based imputation. The embeddings of CVs are individually fed into the imputation decoder to impute missing values of CVs.

## A.7 Pipeline for Long-term Forecasting

Forecasting is a fundamental problem in the time series community, and long-term forecasting is a more practical and challenging task. The long-term forecasting task for MiTS includes the prediction of both DVs and CVs. The overall pipeline of MiTSformer-based long-term forecasting is depicted in Fig. 13. For MiTSformer, the task head is composed of linear layer-based DVforecastor and CVforecaster, which perform variate-wise forecasting for each DV and each CV, respectively. For baseline models, we also adopt the linear layer-based forecaster to forecast the DVs with the extracted temporal features. We adopt the mean absolute error (MAE) and the root mean square error (RMSE) as the forecasting performance evaluation metrics.

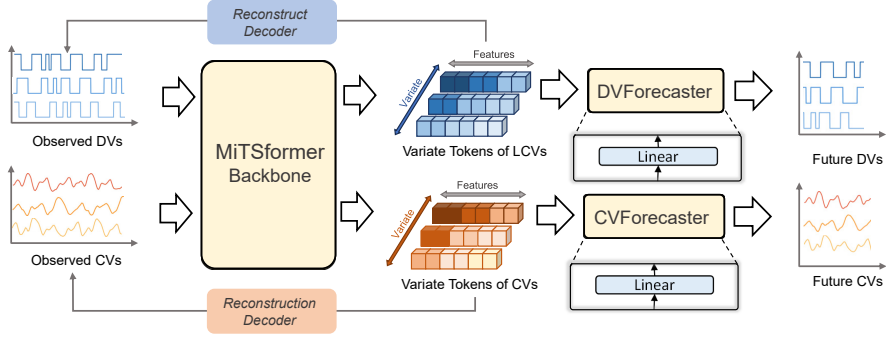

Figure 13: Overall pipeline of MiTSformer-based long-term forecasting. The embeddings of LCVs are individually fed into the DVForecaster to predict the future value of corresponding DVs, and the embeddings of CVs are individually fed into the DVForecaster to predict the future value of corresponding CVs.

## A.8 Pipeline for Anomaly Detection

The anomaly detection task is achieved by self-supervised autoencoding with the reconstruction of both CVs and DVs. The reconstruction errors of DVs and DVs are utilized as anomaly criteria. In our experiment, we estimate the anomaly thresholds respectively for DVs and CVs. For a test sample, the reconstruction error of DVs exceeding the threshold or the reconstruction error of DVs exceeding the threshold is considered an anomaly. The overall pipeline of MiTSformer-based anomaly detection is depicted in Fig. 13. Since there already exist reconstruction decoders and corresponding losses, no task head and additional losses are devised. For baseline models, we also adopt the linear layer-based reconstruct decoders to reconstruct the DVs and linear layer-based reconstruct decoders to reconstruct the CVs with the extracted temporal features. MSE loss is used for reconstruction of CVs and CrossEntropy loss is used for reconstruction of DVs. We adopt the Precision, Recall and F1-Socre as matrices.

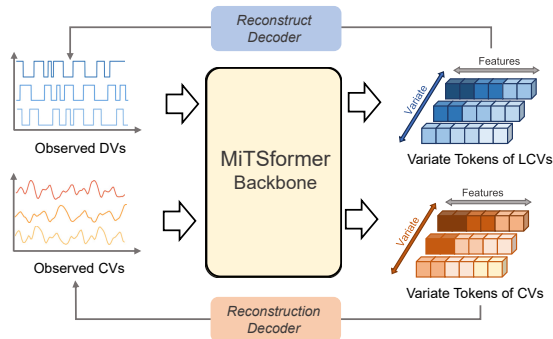

Figure 14: Overall pipeline of MiTSformer-based anomaly detection. The anomaly detection tasks only rely on self-reconstruction and thus no task head is attached.

## B  Model Efficiency Analysis

**Complexity of MiTSformer** Considering the self-attention and cross-attention sub-blocks, the complexity of MiTSformer can be derived as

$$\underbrace{\mathcal{O}\left(n^2\right) + \mathcal{O}\left((p-n)^2\right)}_{\text{self-attention}} + \underbrace{\mathcal{O}\left(n\left(p-n\right)\right) + \mathcal{O}\left((p-n)\,n\right)}_{\text{cross-attention}} \qquad (9)$$

where $n$ denotes the number of DVs and $p - n$ denotes the number of CVs, $p$ is the number of total variables. In our experimental setting, we keep $n = \lfloor 0.5p \rfloor$. Thereby, the complexity of MiTSformer is quadratic to the number of DVs as $\mathcal{O}\left(n^2\right)$.

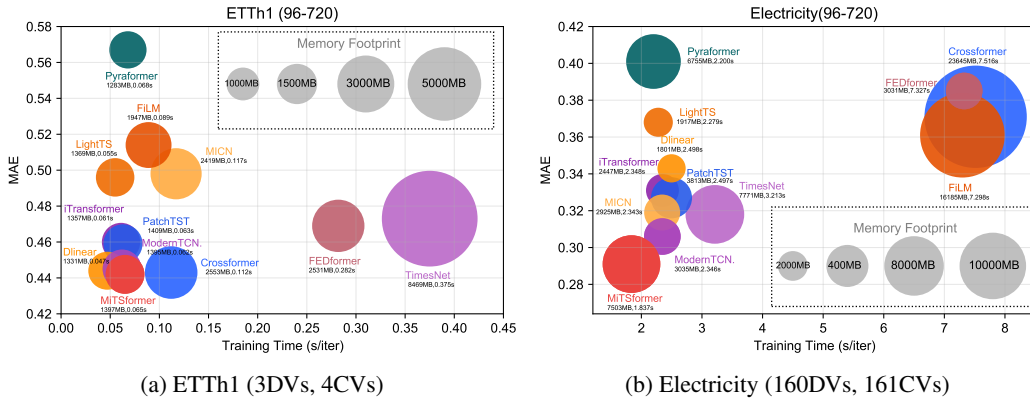

(a) ETTh1 (3DVs, 4CVs)     (b) Electricity (160DVs, 161CVs)

Figure 15: Model efficiency analysis. Experiments are carried out on ETTh1 and Electricity datasets with "input-output" setting of "96-720". For each subfigure, dots with different colors represent different methods, and the size of the circle represents the magnitude of the memory footprint. The horizontal axis represents the training time (seconds) per iter, and the vertical axis represents the forecasting accuracy (MAE).

**Model Efficiency Comparison** We comprehensively compare the model performance, training speed, and memory footprint of MiTSformer and baseline models. The results are with the optimal model configurations as reported in Table 6 and 7. Two representative long-term forecasting datasets ETTh1 (3DVs, 4CVs) and Electricity (160DVs, 161CVs) with input-96 and output-720 settings are adopted for efficiency comparison. The results are depicted in Fig. 15.

In general, MiTSformer maintains great performance and efficiency compared with most baselines in datasets with a relatively small number of variables (ETTh1). When encountering datasets with a relatively large number of variables (Electricity), MiTSformer occupies a relatively large memory footprint, specifically compared with some advanced efficient models such as modern TCN and DLinear. However, the training time of MiTSformer is still efficient.

## C   Hyper-parameter Sensitivity Analysis

We evaluate the hyper-parameter sensitivity of MiTSformer with respect to the following two aspects:

**Sensitivity of Model Capacity** The hidden dimension $d_{model}$ and number of layers $L$ influence the model capacity of MiTSformer. We evaluate the sensitivity of these two hyper-parameters on typical long-term forecasting datasets, including ETTh1, ETTh2, and Weather with four different prediction horizon settings. The results are presented in Fig. 16. We can find that MiTSformer is relatively stable in the selection of $d_{model}$ and $L$, particularly for ETTh2 and Weather datasets. Specifically for the ETTh1 dataset, the model capacities are not essentially favored to be as large as possible.

**Sensitivity of Loss Functions** We further investigate the effects of loss items, including smoothness loss weight $\lambda_1$, reconstruction loss weight $\lambda_2$, and variable modality discrimination loss weight $\lambda_3$ on classification datasets (JapaneseVowels, SpokenArabicDigits, and SelfRegulationSCP1). The results are presented in Fig. 17. In general, we can observe that MiTSformer is quite robust to the weights of loss items, and moderate weights bring optimal performance. For example, $\lambda_1$ controls the magnitude of the smoothing constraints for latent continuity recovery. Too small $\lambda_1$ would make the smoothness invalid while too large $\lambda_1$ may lead to over-smoothing problems. Therefore, a moderate setting of $\lambda_1$ is favored for MiTSforemer.

## D   Limitations and Future Work

**Handling Discrete Variables with Natural Sudden Changes** The smoothness loss in our method is suitable for DVs with sudden changes that are caused by inherent smooth variations. However, it may not adequately account for DVs with inherent sudden changes that are essential characteristics of the dataset. For such cases, we can adjust the coefficient $\lambda_1$ with a relatively small value, or we can

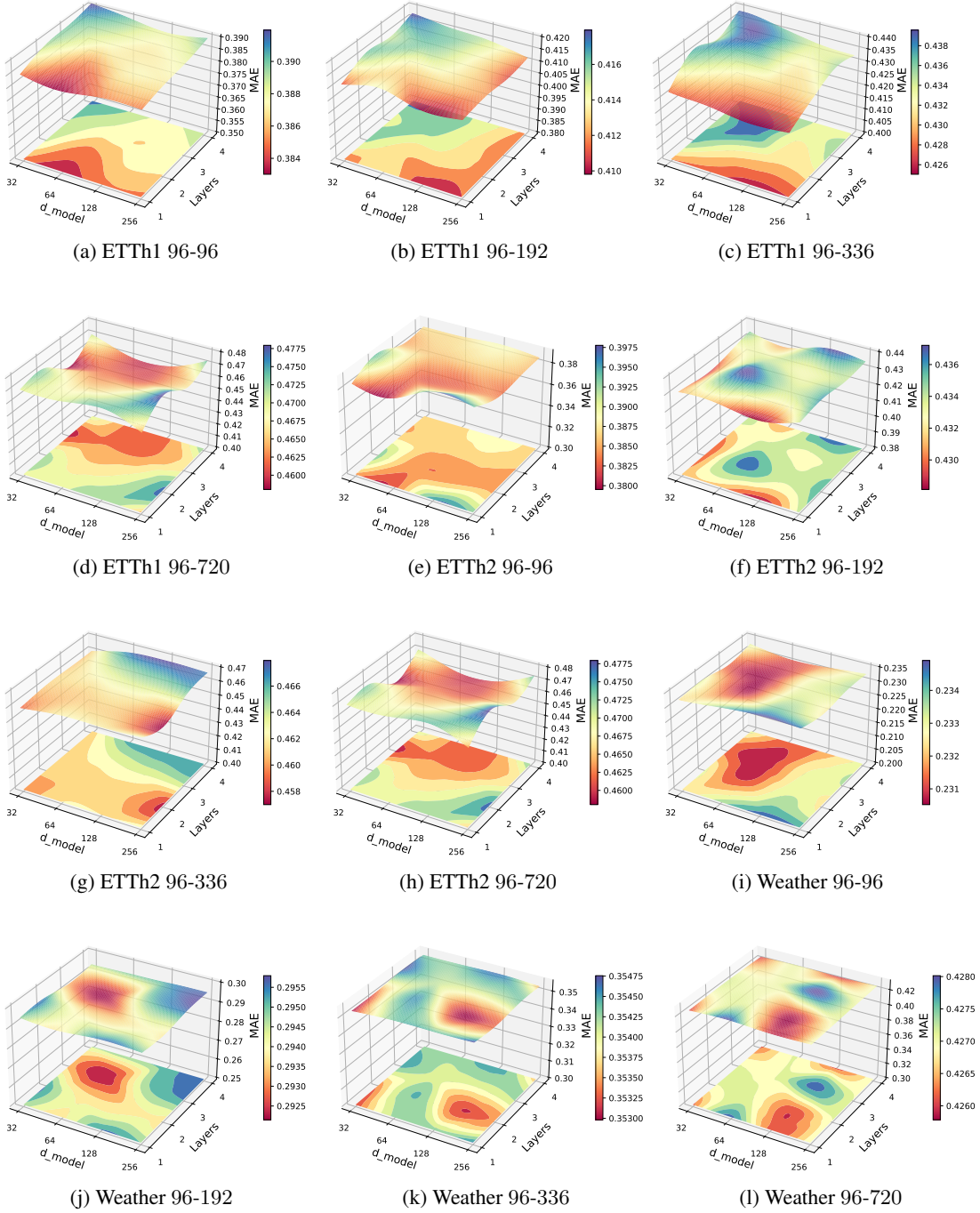

Figure 16: Sensitivity analysis of model capacity-related hyper-parameters $d_{model}$ and number of layers $L$. Experiments are carried out on typical long-term forecasting datasets, including ETTh1, ETTh2, and Weather datasets. "96-96/192/336/720" denotes different forecasting length settings

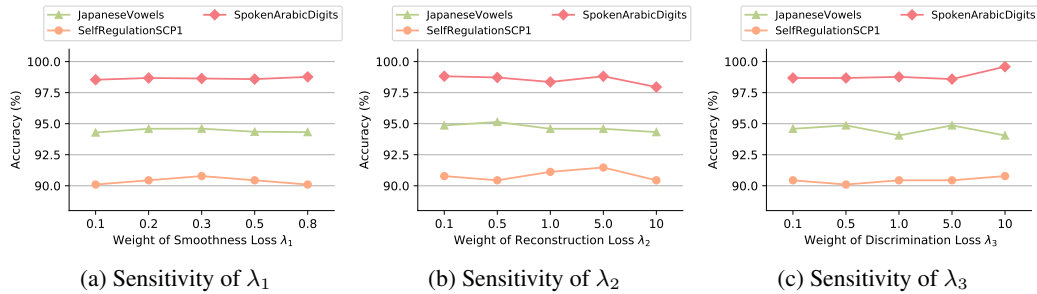

| (a) Sensitivity of $\lambda_1$ | (b) Sensitivity of $\lambda_2$ | (c) Sensitivity of $\lambda_3$ |

Figure 17: Sensitivity analysis of loss items, including smoothness loss weight $\lambda_1$, reconstruction loss weight $\lambda_2$ and variable modality discrimination loss weight $\lambda_3$. Experiments are carried out on classification datasets JapaneseVowels, SpokenArabicDigits, and SelfRegulationSCP1.

further leverage less restrictive constraints such as K-Lipschitz continuity to determine the proper smoothness level.

**Modeling Generalized Mixed Time Series** MiTSformer is based on the assumption that the observed discrete variables are derived from latent continuous variables, which can be guaranteed for *integer* discrete variables whose discrete states represent the magnitude of the numerical value. However, a limitation of MiTSformer is that it can not be directly applied to *categorical* discrete variables whose discrete states represent different and mutually exclusive class labels (e.g., color and gender). It is noted that *integer* discrete variables are more frequently encountered in real-world time series data, specifically in the fields of medical analysis, finance, and industrial processes. In contrast, *categorical* discrete variables are often recorded as independent samples that are not presented in time series forms. In the future, it is of interest to exploit the analysis of generalized variables (including both *integer* discrete variables and *categorical* discrete variables) by leveraging advanced cross-modality modeling and fusion technologies.

**Incorperating Powerful LLMs and Pre-training Techniques** Recently, large language models, such as the Generative Pre-trained Transformer (GPT), have shown impressive performance on various applications and attracted numerous research interests. It is intriguing to explore the potential of GPT-type architectures for mixed time series analysis tasks with proper inductive biases and elaborate prompt strategies. Additionally, by utilizing efficient pre-training techniques, we can enhance the scalability of MiTSformer in large-scale or real-time real-world applications. Meanwhile, the mixed time series analysis on few-shot and zero-shot settings can also be a promising research direction.

# E  Additional Experiments and Discussions

## E.1  Comparison with Mixed Naive Bayes and Variational Inference-based methods

To deal with mixed variables, naive Bayes (NB)-based models [38] and variational inference (VI)-based models [12] are proposed to match the distributions of continuous variables (CVs) and discrete variables (DVs) with different distribution priors for industrial process data analysis. Essentially, mixed NB and VI methods are typically designed for tabular data. They struggle with time series and often rely on certain assumptions like conditional independence, limiting their ability to model the correlations between DVs and CVs. In contrast, MiTSformer leverages temporal adjacencies to achieve latent continuity recovery, making it capable of handling time series data and effectively capturing inherent nonlinear correlations within MiTS. Empirically, we compare MiTSformer against typical mixed NB-based models HVM [1] and VI-based methods VAMDA [2] on mixed time series classification datasets and summarized the results in Table 9, showing MiTSformer consistently outperforms HVM and VAMDA.

## E.2  Investigation of Non-binary Discrete Variable

In our main experimental setting, we chose the binary DVs as they are the most challenging and commonly encountered in the real world. Actually, DVs can take on multiple states ($\geqslant 2$) that reflect the magnitude and can be directly input into the recovery network to obtain LCVs outputs. It is noted

Table 9: Compared to mixed NB- and VI-based methods. Accuracy(%) scores are reported. The best results are **bolded**.

| Datasets | MiTSformer | HVM | VAMDA |
|---|---|---|---|
| Handwriting | **22.6** | 8.1 | 10.8 |
| Heartbeat | **74.6** | 57.7 | 63.4 |
| JapaneseVowels | **94.6** | 86.4 | 78.4 |
| SelfRegulationSCP1 | **91.1** | 85.3 | 88.7 |
| SelfRegulationSCP2 | **60.0** | 51.1 | 54.4 |
| SpokenArabicDigits | **98.5** | 90.2 | 90.8 |

Table 10: Performance of mixed time series classification under the different discrete states of DVs, i.e., $N_{\mathrm{DVs}}$. Accuracy (%) scores are reported. The best results are **bolded**.

| Datasets | $N_{\mathrm{DVs}} = 2$ | $N_{\mathrm{DVs}} = 4$ |
|---|---|---|
| EthanolConcentration | 30.4 | **30.4** |
| FaceDetection | 67.9 | **68.3** |
| Handwriting | 22.6 | **23.1** |
| Heartbeat | **74.6** | 74.1 |
| JapaneseVowels | 94.6 | **95.9** |
| PEMS-SF | **93.1** | 92.5 |
| SelfRegulationSCP1 | 91.1 | **92.8** |
| SelfRegulationSCP2 | 60 | **61.3** |
| SpokenArabicDigits | 98.5 | **98.7** |
| UWaveGestureLibrary | **86.3** | 85.9 |
| Average | 71.9 | **72.3** |

that more states in a DV imply richer information granularity. Also, we conducted experiments on mixed time series classification and anomaly detection datasets with various numbers of discrete states in Tabel 10 and Table 11, respectively. In general, the results shows that as the number of states increases, performance improves due to the richer information.

Table 11: Performance of mixed time series anomaly detection under the different discrete states of DVs, i.e., $N_{\mathrm{DVs}}$.The best results are **bolded**.

| Datasets | Metric | $N_{\mathrm{DVs}} = 2$ | $N_{\mathrm{DVs}} = 4$ |
|---|---|---|---|
| SMD | Precision | 88.92 | 88.37 |
|  | Recall | 86.78 | 87.92 |
|  | F1-score | 87.84 | **88.14** |
| MSL | Precision | 90.66 | 90.81 |
|  | Recall | 80.54 | 81.22 |
|  | F1-score | 85.30 | **85.75** |
| SMAP | Precision | 96.71 | 96.82 |
|  | Recall | 72.21 | 75.26 |
|  | F1-score | 82.69 | **84.69** |
| SWaT | Precision | 96.31 | 94.33 |
|  | Recall | 95.98 | 94.82 |
|  | F1-score | **96.15** | 94.57 |
| PSM | Precision | 97.88 | 98.41 |
|  | Recall | 94.85 | 95.62 |
|  | F1-score | 96.83 | **96.99** |

Table 12: Full classification results. We report the classification accuracy (%) as the result.

| Datasets/Models | MiTS. (Ours) | iTrans. (2024) | M-TCN. (2024) | Times. (2023) | Patch. (2023) | Cross. (2023) | MICN (2023) | Light. (2023) | Dlinear (2023) | FiLM (2022) | FED. (2022) | Pyra. (2022) |
|---|---|---|---|---|---|---|---|---|---|---|---|---|
| EthanolConcentration | 30.4 | 26.6 | 28.5 | 30.0 | 25.9 | 31.9 | 28.9 | 30.4 | 29.3 | 27.8 | 27.0 | 24.8 |
| FaceDetection | 67.9 | 67.9 | 66.1 | 67.9 | 67.5 | 67.3 | 65.2 | 67.9 | 68.0 | 65.9 | 67.4 | 67.4 |
| Handwriting | 22.6 | 17.8 | 18.2 | 15.7 | 13.9 | 15.9 | 6.4 | 8.1 | 11.5 | 8.2 | 14.2 | 17.8 |
| Heartbeat | 74.6 | 66.8 | 76.6 | 73.7 | 72.2 | 72.2 | 72.7 | 72.2 | 74.1 | 72.2 | 73.7 | 74.1 |
| JapaneseVowels | 94.6 | 93.0 | 94.3 | 93.5 | 90.3 | 92.4 | 79.2 | 92.4 | 87.0 | 77.0 | 94.6 | 93.0 |
| PEMS-SF | 93.1 | 87.9 | 84.4 | 86.7 | 83.2 | 83.2 | 85.5 | 85.0 | 86.7 | 85.7 | 85.5 | 84.4 |
| SelfRegulationSCP1 | 91.1 | 87.4 | 92.1 | 89.1 | 86.0 | 89.4 | 89.1 | 90.4 | 89.8 | 84.9 | 73.7 | 89.8 |
| SelfRegulationSCP2 | 60.0 | 58.3 | 57.8 | 56.1 | 56.1 | 57.2 | 56.7 | 56.1 | 57.2 | 55.6 | 55.6 | 54.0 |
| SpokenArabicDigits | 98.5 | 98.4 | 98.1 | 98.7 | 98.0 | 97.7 | 97.7 | 97.8 | 96.1 | 97.3 | 98.7 | 98.9 |
| UWaveGestureLibrary | 86.3 | 84.4 | 84.7 | 84.7 | 83.4 | 82.5 | 82.5 | 81.9 | 80.9 | 74.7 | 57.8 | 84.7 |
| Average | **71.9** | 68.9 | 70.1 | 69.6 | 67.6 | 69.0 | 66.4 | 68.2 | 68.1 | 64.9 | 64.8 | 68.9 |

Table 13: Full extrinsic regression Results. We report the MAE and RMSE as the result.

| Models | MiTSformer (Ours) | | iTransformer (2024) | | modernTCN (2024) | | TimesNet (2023) | | PatchTST (2023) | | Crossformer (2023) | | MICN (2023) | | LightTS (2023) | | Dlinear (2023) | | FiLM (2022) | | FEDformer (2022) | | Pyraformer (2022) | |
|---|---|---|---|---|---|---|---|---|---|---|---|---|---|---|---|---|---|---|---|---|---|---|---|---|
| Metric | MAE | RMSE | MAE | RMSE | MAE | RMSE | MAE | RMSE | MAE | RMSE | MAE | RMSE | MAE | RMSE | MAE | RMSE | MAE | RMSE | MAE | RMSE | MAE | MSE | MAE | RMSE |
| Appli. | 0.579 | 0.730 | 0.642 | 0.864 | 0.605 | 0.805 | 0.664 | 0.850 | 0.675 | 0.877 | 0.707 | 0.946 | 0.641 | 0.808 | 0.706 | 0.873 | 0.692 | 0.892 | 0.751 | 0.970 | 0.676 | 0.896 | 0.654 | 0.853 |
| House.1 | 0.417 | 0.555 | 0.412 | 0.543 | 0.409 | 0.548 | 0.397 | 0.528 | 0.354 | 0.462 | 0.360 | 0.477 | 0.398 | 0.516 | 0.404 | 0.542 | 0.528 | 0.705 | 0.471 | 0.628 | 0.707 | 0.913 | 0.395 | 0.520 |
| House.2 | 0.671 | 0.895 | 0.680 | 0.907 | 0.681 | 0.910 | 0.670 | 0.920 | 0.674 | 0.915 | 0.670 | 0.898 | 0.691 | 0.928 | 0.705 | 0.956 | 0.695 | 0.927 | 0.681 | 0.928 | 0.667 | 0.890 | 0.671 | 0.891 |
| Benze. | 0.352 | 0.481 | 0.371 | 0.518 | 0.390 | 0.515 | 0.364 | 0.475 | 0.357 | 0.492 | 0.352 | 0.484 | 0.395 | 0.526 | 0.368 | 0.486 | 0.359 | 0.477 | 0.541 | 0.745 | 0.450 | 0.591 | 0.391 | 0.494 |
| BJ.PM25 | 0.537 | 0.782 | 0.549 | 0.790 | 0.538 | 0.787 | 0.514 | 0.770 | 0.546 | 0.795 | 0.537 | 0.788 | 0.525 | 0.796 | 0.550 | 0.799 | 0.570 | 0.827 | 0.549 | 0.803 | 0.539 | 0.782 | 0.522 | 0.773 |
| BJ.PM10 | 0.552 | 0.856 | 0.569 | 0.859 | 0.554 | 0.859 | 0.583 | 0.868 | 0.572 | 0.871 | 0.569 | 0.867 | 0.555 | 0.862 | 0.568 | 0.867 | 0.587 | 0.886 | 0.572 | 0.870 | 0.563 | 0.859 | 0.541 | 0.856 |
| Live. | 0.750 | 1.000 | 0.752 | 1.000 | 0.780 | 1.031 | 0.769 | 1.017 | 0.755 | 1.016 | 0.747 | 1.000 | 0.757 | 1.001 | 0.763 | 1.010 | 0.752 | 1.007 | 0.754 | 1.001 | 0.750 | 1.004 | 0.758 | 1.009 |
| Austra. | 0.223 | 0.956 | 0.263 | 0.965 | 0.240 | 0.955 | 0.262 | 0.958 | 0.252 | 0.964 | 0.267 | 0.961 | 0.264 | 0.960 | 0.254 | 0.959 | 0.258 | 0.966 | 0.276 | 0.964 | 0.255 | 0.963 | 0.242 | 0.953 |
| PPGD. | 0.403 | 0.613 | 0.486 | 0.692 | 0.397 | 0.544 | 0.397 | 0.556 | 0.402 | 0.574 | 0.372 | 0.538 | 0.398 | 0.547 | 0.413 | 0.579 | 0.719 | 0.917 | 0.532 | 0.712 | 0.413 | 0.560 | 0.451 | 0.569 |
| IEEE. | 0.755 | 0.928 | 0.779 | 0.973 | 0.755 | 0.939 | 0.906 | 1.063 | 0.751 | 0.945 | 0.765 | 0.941 | 0.736 | 0.932 | 0.811 | 1.030 | 0.906 | 1.073 | 0.851 | 0.996 | 0.741 | 0.902 | 0.776 | 0.945 |
| Average | **0.524** | **0.780** | 0.550 | 0.811 | 0.535 | 0.789 | 0.553 | 0.801 | 0.534 | 0.791 | 0.535 | 0.790 | 0.536 | 0.788 | 0.554 | 0.810 | 0.607 | 0.868 | 0.598 | 0.862 | 0.576 | 0.836 | 0.540 | 0.786 |

Kindly note: For dataset abbreviations,"Appli." denotes "AppliancesEnergy"; "House.1" denotes "Household-PowerConsumption1"; "House.1" denotes "HouseholdPowerConsumption2"; "Benze." denotes "BenzeneConcentration"; "BJ.PM25 " denotes "BeijingPM25Quality"; "BJ.PM10 " denotes "BeijingPM10Quality"; "Live." denotes "LiveFuelMoistureContent"; "Austra." denotes "AustraliaRainfall"; "PPGD." denotes "PPGDalia"; and "IEEE." denotes "IEEEPPG".

# F   Full Experimental Results

The full results of all experiments are presented in the following: classification in Table 12, extrinsic regression in Table 13, imputation in Table 14, anomaly detection in Table 15. Besides, long-term forecasting results of CVs are presented in Table 16 and those of DVs are presented in 17.

Table 14: Full imputation results. We randomly mask 12.5%, 25%, 37.5% and 50% time points to compare the model performance under different missing degrees.

| Models | | MiTSformer (Ours) | | iTransformer (2024) | | M-TCN (2024) | | TimesNet (2023) | | PatchTST (2023) | | Crossformer (2023) | | MICN (2023) | | LightTS (2023) | | Dlinear (2023) | | FiLM (2022) | | FEDformer (2022) | | Pyraformer (2022) | |
|---|---|---|---|---|---|---|---|---|---|---|---|---|---|---|---|---|---|---|---|---|---|---|---|---|
| | M-Ratio | MAE | MSE | MAE | MSE | MAE | MSE | MAE | MSE | MAE | MSE | MAE | MSE | MAE | MSE | MAE | MSE | MAE | MSE | MAE | MSE | MAE | MSE | MAE | MSE |
| ETTm1 | 12.5% | 0.143 | 0.042 | 0.149 | 0.046 | 0.123 | 0.031 | 0.119 | 0.029 | 0.151 | 0.048 | 0.150 | 0.045 | 0.137 | 0.038 | 0.156 | 0.050 | 0.159 | 0.053 | 0.161 | 0.054 | 0.144 | 0.046 | 0.178 | 0.063 |
| | 25% | 0.152 | 0.047 | 0.162 | 0.053 | 0.130 | 0.034 | 0.131 | 0.035 | 0.159 | 0.053 | 0.155 | 0.048 | 0.156 | 0.049 | 0.172 | 0.058 | 0.182 | 0.069 | 0.183 | 0.070 | 0.161 | 0.055 | 0.192 | 0.080 |
| | 37.5% | 0.160 | 0.051 | 0.175 | 0.060 | 0.138 | 0.038 | 0.140 | 0.040 | 0.167 | 0.058 | 0.162 | 0.051 | 0.169 | 0.057 | 0.185 | 0.068 | 0.203 | 0.087 | 0.204 | 0.087 | 0.172 | 0.059 | 0.204 | 0.085 |
| | 50% | 0.170 | 0.057 | 0.190 | 0.070 | 0.149 | 0.043 | 0.164 | 0.052 | 0.180 | 0.067 | 0.172 | 0.058 | 0.178 | 0.065 | 0.200 | 0.079 | 0.226 | 0.110 | 0.227 | 0.110 | 0.202 | 0.079 | 0.222 | 0.096 |
| | Avg. | 0.156 | 0.049 | 0.169 | 0.057 | 0.135 | 0.037 | 0.139 | 0.039 | 0.164 | 0.057 | 0.160 | 0.051 | 0.160 | 0.052 | 0.178 | 0.064 | 0.193 | 0.080 | 0.194 | 0.080 | 0.170 | 0.060 | 0.199 | 0.081 |
| ETTm2 | 12.5% | 0.104 | 0.030 | 0.163 | 0.062 | 0.169 | 0.064 | 0.134 | 0.040 | 0.132 | 0.046 | 0.194 | 0.080 | 0.174 | 0.066 | 0.197 | 0.087 | 0.196 | 0.086 | 0.211 | 0.100 | 0.178 | 0.065 | 0.234 | 0.105 |
| | 25% | 0.112 | 0.034 | 0.182 | 0.076 | 0.178 | 0.068 | 0.159 | 0.055 | 0.138 | 0.050 | 0.211 | 0.091 | 0.224 | 0.104 | 0.205 | 0.093 | 0.245 | 0.135 | 0.246 | 0.136 | 0.211 | 0.092 | 0.246 | 0.121 |
| | 37.5% | 0.120 | 0.038 | 0.195 | 0.087 | 0.193 | 0.079 | 0.180 | 0.072 | 0.147 | 0.056 | 0.225 | 0.113 | 0.268 | 0.151 | 0.213 | 0.098 | 0.272 | 0.165 | 0.274 | 0.169 | 0.258 | 0.140 | 0.279 | 0.159 |
| | 50% | 0.128 | 0.043 | 0.205 | 0.096 | 0.212 | 0.095 | 0.205 | 0.092 | 0.162 | 0.066 | 0.232 | 0.128 | 0.313 | 0.203 | 0.219 | 0.102 | 0.300 | 0.200 | 0.302 | 0.203 | 0.303 | 0.193 | 0.302 | 0.179 |
| | Avg. | 0.116 | 0.036 | 0.186 | 0.080 | 0.188 | 0.077 | 0.170 | 0.065 | 0.145 | 0.055 | 0.216 | 0.103 | 0.245 | 0.131 | 0.209 | 0.095 | 0.253 | 0.147 | 0.258 | 0.152 | 0.238 | 0.123 | 0.265 | 0.141 |
| ETTh1 | 12.5% | 0.203 | 0.080 | 0.213 | 0.091 | 0.190 | 0.070 | 0.201 | 0.078 | 0.221 | 0.100 | 0.225 | 0.104 | 0.196 | .077 | 0.239 | 0.114 | 0.223 | 0.103 | 0.231 | 0.109 | .201 | 0.075 | 0.214 | 0.085 |
| | 25% | 0.217 | 0.091 | 0.232 | 0.108 | 0.204 | 0.077 | 0.222 | 0.097 | 0.241 | 0.118 | 0.240 | 0.116 | 0.221 | 0.095 | 0.254 | 0.131 | 0.250 | 0.130 | 0.256 | 0.135 | 0.226 | 0.095 | 0.227 | 0.095 |
| | 37.5% | 0.230 | 0.100 | 0.250 | 0.124 | 0.223 | 0.093 | 0.247 | 0.119 | 0.259 | 0.136 | 0.251 | 0.126 | 0.247 | 0.115 | 0.274 | 0.155 | 0.275 | 0.156 | 0.279 | 0.165 | 0.257 | 0.122 | 0.254 | 0.118 |
| | 50% | 0.243 | 0.111 | 0.269 | 0.142 | 0.244 | 0.126 | 0.279 | 0.152 | 0.281 | 0.163 | 0.266 | 0.139 | 0.272 | 0.139 | 0.296 | 0.181 | 0.300 | 0.192 | 0.304 | 0.198 | 0.292 | 0.157 | 0.291 | 0.152 |
| | Avg. | 0.223 | 0.096 | 0.241 | 0.116 | 0.215 | 0.092 | 0.237 | 0.112 | 0.251 | 0.129 | 0.246 | 0.121 | 0.234 | 0.107 | 0.266 | 0.145 | 0.262 | 0.145 | 0.268 | 0.152 | 0.244 | 0.112 | 0.247 | 0.113 |
| ETTh2 | 12.5% | 0.175 | 0.073 | 0.224 | 0.109 | 0.259 | 0.148 | 0.234 | 0.120 | 0.237 | 0.122 | 0.277 | 0.163 | 0.246 | 0.129 | 0.290 | 0.180 | 0.255 | 0.140 | 0.271 | 0.163 | 0.279 | 0.158 | 0.328 | 0.195 |
| | 25% | 0.182 | 0.079 | 0.242 | 0.125 | 0.293 | 0.186 | 0.300 | 0.201 | 0.253 | 0.140 | 0.303 | 0.195 | 0.286 | 0.170 | 0.308 | 0.202 | 0.284 | 0.168 | 0.317 | 0.220 | 0.325 | 0.209 | 0.342 | 0.226 |
| | 37.5% | 0.189 | 0.085 | 0.258 | 0.141 | 0.325 | 0.230 | 0.332 | 0.243 | 0.268 | 0.158 | 0.311 | 0.206 | 0.338 | 0.239 | 0.322 | 0.220 | 0.340 | 0.246 | 0.365 | 0.291 | 0.370 | 0.276 | 0.383 | 0.269 |
| | 50% | 0.199 | 0.093 | 0.276 | 0.160 | 0.360 | 0.287 | 0.408 | 0.390 | 0.282 | 0.173 | 0.325 | 0.226 | 0.403 | 0.343 | 0.335 | 0.235 | 0.368 | 0.289 | 0.413 | 0.372 | 0.424 | 0.390 | 0.415 | 0.320 |
| | Avg. | 0.186 | 0.083 | 0.250 | 0.134 | 0.309 | 0.213 | 0.319 | 0.239 | 0.260 | 0.148 | 0.304 | 0.198 | 0.318 | 0.220 | 0.314 | 0.209 | 0.312 | 0.211 | 0.342 | 0.262 | 0.350 | 0.258 | 0.367 | 0.253 |
| Electricity | 12.5% | 0.154 | 0.055 | 0.184 | 0.071 | 0.178 | 0.065 | 0.189 | 0.075 | 0.180 | 0.067 | 0.181 | 0.066 | 0.198 | 0.077 | 0.191 | 0.075 | 0.206 | 0.086 | 0.206 | 0.085 | 0.221 | 0.093 | 0.259 | 0.130 |
| | 25% | 0.177 | 0.069 | 0.204 | 0.086 | 0.200 | 0.082 | 0.204 | 0.087 | 0.194 | 0.079 | 0.193 | 0.075 | 0.220 | 0.093 | 0.214 | 0.091 | 0.243 | 0.115 | 0.243 | 0.115 | 0.250 | 0.117 | 0.271 | 0.143 |
| | 37.5% | 0.195 | 0.082 | 0.221 | 0.099 | 0.214 | 0.093 | 0.216 | 0.099 | 0.207 | 0.091 | 0.205 | 0.117 | 0.240 | 0.111 | 0.233 | 0.106 | 0.273 | 0.144 | 0.274 | 0.134 | 0.276 | 0.142 | 0.279 | 0.157 |
| | 50% | 0.216 | 0.097 | 0.239 | 0.114 | 0.236 | 0.113 | 0.233 | 0.115 | 0.231 | 0.222 | 0.257 | 0.131 | 0.259 | 0.131 | 0.252 | 0.122 | 0.304 | 0.175 | 0.304 | 0.176 | 0.292 | 0.169 | 0.288 | 0.169 |
| | Avg. | 0.186 | 0.076 | 0.212 | 0.093 | 0.207 | 0.088 | 0.211 | 0.094 | 0.203 | 0.115 | 0.209 | 0.097 | 0.229 | 0.103 | 0.223 | 0.099 | 0.257 | 0.130 | 0.257 | 0.128 | 0.260 | 0.130 | 0.274 | 0.150 |
| Weather | 12.5% | 0.055 | 0.029 | 0.080 | 0.033 | 0.074 | 0.030 | 0.093 | 0.035 | 0.083 | 0.034 | 0.116 | 0.044 | 0.094 | 0.035 | 0.095 | 0.036 | 0.097 | 0.037 | 0.100 | 0.038 | 0.103 | 0.039 | 0.082 | 0.038 |
| | 25% | 0.060 | 0.030 | 0.088 | 0.036 | 0.081 | 0.032 | 0.136 | 0.051 | 0.085 | 0.035 | 0.111 | 0.041 | 0.122 | 0.047 | 0.107 | 0.040 | 0.116 | 0.045 | 0.118 | 0.046 | 0.119 | 0.045 | 0.093 | 0.041 |
| | 37.5% | 0.065 | 0.032 | 0.095 | 0.039 | 0.083 | 0.033 | 0.173 | 0.074 | 0.088 | 0.037 | 0.114 | 0.043 | 0.148 | 0.060 | 0.106 | 0.041 | 0.129 | 0.051 | 0.131 | 0.052 | 0.144 | 0.058 | 0.105 | 0.042 |
| | 50% | 0.069 | 0.034 | 0.102 | 0.042 | 0.087 | 0.035 | 0.195 | 0.098 | 0.097 | 0.041 | 0.119 | 0.045 | 0.176 | 0.076 | 0.118 | 0.046 | 0.143 | 0.059 | 0.145 | 0.600 | 0.188 | 0.086 | 0.118 | 0.047 |
| | Avg. | 0.062 | 0.031 | 0.091 | 0.038 | 0.081 | 0.033 | 0.149 | 0.065 | 0.088 | 0.037 | 0.115 | 0.043 | 0.135 | 0.055 | 0.107 | 0.041 | 0.121 | 0.048 | 0.124 | 0.184 | 0.139 | 0.057 | 0.100 | 0.042 |
| Overall Avg. | | 0.155 | 0.062 | 0.192 | 0.086 | 0.189 | 0.090 | 0.204 | 0.102 | 0.185 | 0.090 | 0.208 | 0.102 | 0.220 | 0.111 | 0.216 | 0.109 | 0.233 | 0.127 | 0.240 | 0.160 | 0.233 | 0.123 | 0.242 | 0.130 |
| Top1 Counts. | | 34 | | 0 | | 12 | | 2 | | 0 | | 0 | | 0 | | 0 | | 0 | | 1 | | 0 | | 0 | |

Table 15: Full anomaly detection results. The "P", "R", and "F1" represent the precision, recall, and F1-score (%) respectively. F1-score is the harmonic mean of precision and recall. A higher value of P, R, and F1 indicates better anomaly detection performance. .

| Dataset | SMD | | | MSL | | | SMAP | | | SWaT | | | PSM | | | Avg. F1 (%) |
|---|---|---|---|---|---|---|---|---|---|---|---|---|---|---|---|---|
| Metrics | P | R | F1 | P | R | F1 | P | R | F1 | P | R | F1 | P | R | F1 | |
| **MiTSformer** | 88.92 | 86.78 | 87.84 | 90.66 | 80.54 | 85.30 | 96.71 | 72.21 | 82.69 | 96.31 | 95.98 | 96.15 | 97.88 | 94.85 | 96.83 | 89.76 |
| iTransformer(2024) | 87.66 | 75.90 | 81.36 | 84.13 | 82.87 | 83.50 | 86.72 | 77.15 | 81.65 | 96.24 | 93.77 | 94.99 | 98.88 | 92.87 | 95.78 | 87.46 |
| ModernTCN(2024) | 86.90 | 73.95 | 79.91 | 93.25 | 60.50 | 73.39 | 96.84 | 50.69 | 66.50 | 96.34 | 94.45 | 95.39 | 98.92 | 93.20 | 95.98 | 82.23 |
| TimesNet(2023) | 88.56 | 81.32 | 84.78 | 94.41 | 71.01 | 81.06 | 95.04 | 54.50 | 69.27 | 95.69 | 95.74 | 95.71 | 98.56 | 92.49 | 95.43 | 85.25 |
| PatchTST(2023) | 87.75 | 75.35 | 81.08 | 94.83 | 67.27 | 78.71 | 94.51 | 54.85 | 69.53 | 96.28 | 95.16 | 95.72 | 99.15 | 92.70 | 95.82 | 84.17 |
| Crossformer(2023) | 83.98 | 75.76 | 79.60 | 95.73 | 69.85 | 80.77 | 95.21 | 60.07 | 73.67 | 94.47 | 94.77 | 96.11 | 98.39 | 92.89 | 95.57 | 85.14 |
| MICN(2023) | 89.65 | 82.47 | 85.91 | 95.35 | 71.42 | 81.67 | 95.01 | 56.62 | 67.73 | 95.25 | 95.72 | 95.48 | 98.92 | 88.22 | 93.27 | 84.81 |
| LightTS (2023) | 87.10 | 70.73 | 76.07 | 94.85 | 69.57 | 80.27 | 94.77 | 52.94 | 67.93 | 96.24 | 95.75 | 96.00 | 98.58 | 92.51 | 95.45 | 83.14 |
| Dlinear(2023) | 87.91 | 78.86 | 83.14 | 94.40 | 60.45 | 73.71 | 95.51 | 52.33 | 67.36 | 97.24 | 94.88 | 96.04 | 98.99 | 92.80 | 95.79 | 83.21 |
| FiLM (2022) | 87.07 | 75.46 | 80.85 | 74.62 | 59.42 | 73.00 | 94.82 | 51.20 | 66.53 | 96.17 | 95.48 | 95.83 | 98.53 | 92.45 | 95.39 | 82.32 |
| FEDformer (2022) | 85.23 | 84.14 | 84.68 | 96.05 | 68.77 | 80.15 | 95.35 | 54.82 | 69.61 | 96.99 | 94.31 | 95.63 | 99.11 | 92.08 | 95.47 | 85.11 |
| Pyraformer(2022) | 83.27 | 71.55 | 76.97 | 96.07 | 71.99 | 82.31 | 95.53 | 62.08 | 75.26 | 99.05 | 92.81 | 95.83 | 98.85 | 91.46 | 95.01 | 85.08 |

Table 16: Full long-term forecasting results for CVs. We compare extensive competitive models under different prediction lengths. The input sequence length is set to 36 for the ILI dataset and 96 for the others. Avg. is averaged from all four prediction lengths. The performance is evaluated by MAE and RMSE. "-" denotes "out of memory".

| Models | Length | MiTSformer (Ours) MAE | MSE | iTransformer (2024) MAE | MSE | M-TCN (2024) MAE | MSE | TimesNet (2023) MAE | MSE | PatchTST (2023) MAE | MSE | Crossformer (2023) MAE | MSE | MICN (2023) MAE | MSE | LightTS (2023) MAE | MSE | Dlinear (2023) MAE | MSE | FiLM (2022) MAE | MSE | FEDformer (2022) MAE | MSE | Pyraformer (2022) MAE | MSE |
|---|---|---|---|---|---|---|---|---|---|---|---|---|---|---|---|---|---|---|---|---|---|---|---|---|---|
| ETTm1 | 96 | 0.342 | 0.271 | 0.348 | 0.279 | 0.346 | 0.272 | 0.363 | 0.288 | 0.346 | 0.276 | 0.368 | 0.296 | 0.352 | 0.275 | 0.364 | 0.296 | 0.344 | 0.275 | 0.362 | 0.297 | 0.398 | 0.319 | 0.443 | 0.408 |
| | 192 | 0.363 | 0.310 | 0.371 | 0.321 | 0.367 | 0.314 | 0.400 | 0.351 | 0.367 | 0.313 | 0.401 | 0.349 | 0.369 | 0.321 | 0.382 | 0.331 | 0.368 | 0.312 | 0.392 | 0.348 | 0.415 | 0.356 | 0.467 | 0.441 |
| | 336 | 0.382 | 0.338 | 0.391 | 0.347 | 0.387 | 0.344 | 0.413 | 0.371 | 0.387 | 0.342 | 0.416 | 0.368 | 0.389 | 0.328 | 0.401 | 0.358 | 0.388 | 0.336 | 0.404 | 0.367 | 0.436 | 0.393 | 0.492 | 0.479 |
| | 720 | 0.415 | 0.391 | 0.430 | 0.414 | 0.421 | 0.404 | 0.437 | 0.416 | 0.416 | 0.389 | 0.442 | 0.417 | 0.429 | 0.399 | 0.430 | 0.407 | 0.423 | 0.399 | 0.428 | 0.414 | 0.451 | 0.414 | 0.532 | 0.552 |
| | Avg. | 0.376 | 0.328 | 0.385 | 0.340 | 0.380 | 0.334 | 0.403 | 0.357 | 0.379 | 0.330 | 0.407 | 0.358 | 0.385 | 0.331 | 0.394 | 0.348 | 0.381 | 0.331 | 0.397 | 0.357 | 0.425 | 0.371 | 0.484 | 0.470 |
| ETTm2 | 96 | 0.293 | 0.231 | 0.298 | 0.241 | 0.297 | 0.239 | 0.300 | 0.250 | 0.299 | 0.232 | 0.434 | 0.397 | 0.352 | 0.277 | 0.388 | 0.312 | 0.376 | 0.304 | 0.305 | 0.256 | 0.320 | 0.255 | 0.459 | 0.381 |
| | 192 | 0.344 | 0.324 | 0.351 | 0.331 | 0.339 | 0.319 | 0.345 | 0.334 | 0.340 | 0.316 | 0.566 | 0.649 | 0.480 | 0.484 | 0.481 | 0.473 | 0.443 | 0.421 | 0.357 | 0.345 | 0.365 | 0.349 | 0.581 | 0.582 |
| | 336 | 0.383 | 0.395 | 0.388 | 0.402 | 0.385 | 0.400 | 0.400 | 0.437 | 0.390 | 0.399 | 0.735 | 0.970 | 0.575 | 0.670 | 0.555 | 0.614 | 0.539 | 0.604 | 0.394 | 0.418 | 0.403 | 0.417 | 0.856 | 1.195 |
| | 720 | 0.439 | 0.503 | 0.445 | 0.518 | 0.444 | 0.525 | 0.459 | 0.558 | 0.443 | 0.509 | 1.776 | 2.209 | 0.703 | 0.963 | 0.690 | 0.913 | 0.688 | 0.943 | 0.449 | 0.538 | 0.457 | 0.519 | 1.688 | 4.769 |
| | Avg. | 0.365 | 0.363 | 0.371 | 0.373 | 0.366 | 0.371 | 0.376 | 0.395 | 0.368 | 0.364 | 0.878 | 1.056 | 0.528 | 0.599 | 0.529 | 0.578 | 0.512 | 0.568 | 0.376 | 0.389 | 0.386 | 0.385 | 0.896 | 1.732 |
| ETTh1 | 96 | 0.381 | 0.323 | 0.393 | 0.338 | 0.382 | 0.335 | 0.411 | 0.354 | 0.382 | 0.330 | 0.392 | 0.327 | 0.390 | 0.328 | 0.451 | 0.426 | 0.385 | 0.331 | 0.397 | 0.365 | 0.401 | 0.335 | 0.483 | 0.483 |
| | 192 | 0.408 | 0.369 | 0.419 | 0.383 | 0.408 | 0.381 | 0.434 | 0.390 | 0.404 | 0.371 | 0.418 | 0.371 | 0.426 | 0.374 | 0.475 | 0.462 | 0.413 | 0.375 | 0.419 | 0.399 | 0.430 | 0.373 | 0.518 | 0.533 |
| | 336 | 0.424 | 0.402 | 0.436 | 0.419 | 0.426 | 0.417 | 0.464 | 0.446 | 0.417 | 0.408 | 0.446 | 0.418 | 0.476 | 0.442 | 0.488 | 0.491 | 0.427 | 0.404 | 0.443 | 0.441 | 0.449 | 0.416 | 0.535 | 0.563 |
| | 720 | 0.442 | 0.398 | 0.460 | 0.430 | 0.445 | 0.418 | 0.473 | 0.459 | 0.459 | 0.416 | 0.443 | 0.389 | 0.498 | 0.472 | 0.496 | 0.479 | 0.444 | 0.395 | 0.514 | 0.516 | 0.469 | 0.434 | 0.567 | 0.591 |
| | Avg. | 0.414 | 0.373 | 0.427 | 0.393 | 0.415 | 0.388 | 0.446 | 0.412 | 0.416 | 0.381 | 0.425 | 0.376 | 0.448 | 0.404 | 0.478 | 0.465 | 0.417 | 0.376 | 0.443 | 0.430 | 0.437 | 0.390 | 0.526 | 0.543 |
| ETTh2 | 96 | 0.380 | 0.369 | 0.386 | 0.380 | 0.395 | 0.402 | 0.428 | 0.429 | 0.391 | 0.388 | 0.737 | 1.016 | 0.504 | 0.546 | 0.555 | 0.622 | 0.394 | 0.382 | 0.404 | 0.411 | 0.426 | 0.429 | 1.014 | 1.778 |
| | 192 | 0.430 | 0.458 | 0.438 | 0.482 | 0.442 | 0.490 | 0.442 | 0.494 | 0.437 | 0.460 | 0.899 | 1.447 | 0.612 | 0.775 | 0.636 | 0.795 | 0.621 | 0.777 | 0.444 | 0.496 | 0.471 | 0.515 | 1.326 | 2.645 |
| | 336 | 0.455 | 0.494 | 0.464 | 0.508 | 0.454 | 0.506 | 0.468 | 0.529 | 0.457 | 0.495 | 0.944 | 1.430 | 0.723 | 1.028 | 0.765 | 1.113 | 0.736 | 1.067 | 0.463 | 0.518 | 0.499 | 0.546 | 1.414 | 2.819 |
| | 720 | 0.455 | 0.473 | 0.478 | 0.519 | 0.478 | 0.523 | 0.472 | 0.509 | 0.476 | 0.511 | 1.088 | 1.795 | 0.930 | 1.651 | 0.960 | 1.720 | 0.948 | 1.701 | 0.487 | 0.536 | 0.507 | 0.542 | 1.462 | 2.951 |
| | Avg. | 0.430 | 0.449 | 0.442 | 0.472 | 0.442 | 0.480 | 0.453 | 0.490 | 0.440 | 0.464 | 0.917 | 1.422 | 0.692 | 1.000 | 0.729 | 1.063 | 0.675 | 0.982 | 0.450 | 0.490 | 0.476 | 0.508 | 1.304 | 2.548 |
| Weather | 96 | 0.231 | 0.147 | 0.243 | 0.160 | 0.232 | 0.140 | 0.242 | 0.152 | 0.247 | 0.163 | 0.244 | 0.141 | 0.283 | 0.180 | 0.269 | 0.171 | 0.267 | 0.173 | 0.266 | 0.183 | 0.309 | 0.205 | 0.259 | 0.152 |
| | 192 | 0.294 | 0.216 | 0.302 | 0.228 | 0.292 | 0.218 | 0.307 | 0.230 | 0.297 | 0.220 | 0.301 | 0.201 | 0.327 | 0.233 | 0.319 | 0.229 | 0.318 | 0.228 | 0.317 | 0.245 | 0.365 | 0.279 | 0.334 | 0.228 |
| | 336 | 0.353 | 0.298 | 0.361 | 0.310 | 0.355 | 0.299 | 0.374 | 0.322 | 0.356 | 0.303 | 0.357 | 0.272 | 0.379 | 0.304 | 0.381 | 0.304 | 0.366 | 0.309 | 0.366 | 0.318 | 0.405 | 0.347 | 0.390 | 0.310 |
| | 720 | 0.426 | 0.412 | 0.429 | 0.417 | 0.431 | 0.420 | 0.467 | 0.479 | 0.428 | 0.418 | 0.448 | 0.415 | 0.435 | 0.394 | 0.447 | 0.402 | 0.432 | 0.384 | 0.461 | 0.418 | 0.465 | 0.457 | 0.446 | 0.405 |
| | Avg. | 0.326 | 0.268 | 0.334 | 0.279 | 0.328 | 0.269 | 0.348 | 0.296 | 0.332 | 0.276 | 0.338 | 0.257 | 0.356 | 0.278 | 0.354 | 0.277 | 0.346 | 0.274 | 0.353 | 0.291 | 0.386 | 0.322 | 0.357 | 0.274 |
| Exchange | 96 | 0.221 | 0.094 | 0.227 | 0.098 | 0.225 | 0.097 | 0.266 | 0.131 | 0.224 | 0.095 | 0.345 | 0.225 | 0.236 | 0.101 | 0.264 | 0.120 | 0.238 | 0.103 | 0.251 | 0.112 | 0.324 | 0.187 | 0.481 | 0.390 |
| | 192 | 0.327 | 0.194 | 0.335 | 0.211 | 0.333 | 0.205 | 0.369 | 0.246 | 0.341 | 0.219 | 0.503 | 0.443 | 0.339 | 0.198 | 0.384 | 0.248 | 0.336 | 0.198 | 0.340 | 0.213 | 0.430 | 0.325 | 0.577 | 0.528 |
| | 336 | 0.445 | 0.339 | 0.455 | 0.366 | 0.460 | 0.369 | 0.501 | 0.429 | 0.452 | 0.360 | 0.690 | 0.762 | 0.432 | 0.311 | 0.473 | 0.370 | 0.433 | 0.321 | 0.447 | 0.358 | 0.552 | 0.519 | 0.721 | 0.815 |
| | 720 | 0.788 | 0.963 | 0.790 | 0.974 | 0.774 | 0.942 | 0.876 | 1.186 | 0.795 | 0.995 | 0.844 | 1.099 | 0.640 | 0.683 | 0.714 | 0.869 | 0.629 | 0.651 | 0.757 | 0.910 | 0.948 | 1.365 | 0.819 | 0.982 |
| | Avg. | 0.445 | 0.398 | 0.452 | 0.412 | 0.448 | 0.403 | 0.503 | 0.498 | 0.453 | 0.417 | 0.596 | 0.632 | 0.412 | 0.323 | 0.459 | 0.402 | 0.409 | 0.318 | 0.449 | 0.398 | 0.564 | 0.599 | 0.650 | 0.679 |
| ILI | 24 | 0.771 | 1.542 | 0.950 | 2.112 | 0.938 | 2.060 | 0.882 | 1.918 | 0.950 | 2.061 | 1.079 | 2.744 | 2.211 | 2.587 | 1.688 | 5.046 | 1.456 | 3.552 | 1.123 | 2.642 | 1.377 | 3.322 | 1.101 | 2.533 |
| | 36 | 0.761 | 1.387 | 0.961 | 2.066 | 0.933 | 1.966 | 0.886 | 1.994 | 0.939 | 2.046 | 1.100 | 2.852 | 1.047 | 2.155 | 1.696 | 5.217 | 1.351 | 3.231 | 1.208 | 2.621 | 1.275 | 3.056 | 1.063 | 2.745 |
| | 48 | 0.777 | 1.482 | 0.962 | 2.040 | 0.907 | 1.904 | 0.898 | 2.078 | 0.985 | 2.183 | 1.158 | 3.014 | 1.055 | 2.203 | 1.749 | 5.574 | 1.273 | 2.969 | 1.090 | 2.664 | 1.207 | 2.826 | 1.104 | 2.832 |
| | 60 | 0.805 | 1.516 | 1.105 | 2.309 | 0.911 | 1.897 | 0.897 | 2.071 | 1.017 | 2.269 | 1.222 | 3.238 | 1.117 | 2.496 | 1.804 | 5.890 | 1.278 | 3.034 | 1.330 | 2.881 | 1.208 | 2.806 | 1.115 | 2.877 |
| | Avg. | 0.779 | 1.482 | 0.995 | 2.132 | 0.922 | 1.957 | 0.891 | 2.015 | 0.973 | 2.140 | 1.140 | 2.962 | 1.358 | 2.360 | 1.734 | 5.432 | 1.340 | 3.197 | 1.188 | 2.702 | 1.267 | 3.003 | 1.096 | 2.747 |
| Electricity | 96 | 0.235 | 0.143 | 0.268 | 0.180 | 0.243 | 0.145 | 0.263 | 0.161 | 0.275 | 0.187 | 0.268 | 0.171 | 0.278 | 0.167 | 0.318 | 0.218 | 0.297 | 0.207 | 0.286 | 0.207 | 0.330 | 0.220 | 0.390 | 0.314 |
| | 192 | 0.250 | 0.159 | 0.278 | 0.189 | 0.259 | 0.165 | 0.279 | 0.179 | 0.280 | 0.190 | 0.315 | 0.226 | 0.286 | 0.176 | 0.326 | 0.225 | 0.300 | 0.216 | 0.305 | 0.216 | 0.336 | 0.227 | 0.401 | 0.319 |
| | 336 | 0.265 | 0.171 | 0.296 | 0.207 | 0.272 | 0.174 | 0.290 | 0.189 | 0.296 | 0.204 | 0.349 | 0.257 | 0.298 | 0.186 | 0.342 | 0.240 | 0.314 | 0.217 | 0.313 | 0.227 | 0.352 | 0.247 | 0.407 | 0.326 |
| | 720 | 0.291 | 0.198 | 0.331 | 0.252 | 0.306 | 0.210 | 0.318 | 0.217 | 0.327 | 0.245 | 0.371 | 0.292 | 0.319 | 0.209 | 0.368 | 0.276 | 0.343 | 0.251 | 0.361 | 0.284 | 0.385 | 0.296 | 0.401 | 0.315 |
| | Avg. | 0.260 | 0.168 | 0.293 | 0.207 | 0.270 | 0.174 | 0.288 | 0.187 | 0.295 | 0.207 | 0.326 | 0.237 | 0.295 | 0.185 | 0.339 | 0.240 | 0.314 | 0.220 | 0.316 | 0.234 | 0.351 | 0.248 | 0.400 | 0.319 |
| Traffic | 96 | 0.292 | 0.459 | 0.357 | 0.560 | 0.358 | 0.597 | 0.333 | 0.762 | 0.350 | 0.575 | - | - | 0.333 | 0.652 | 0.464 | 0.831 | 0.432 | 0.774 | - | - | 0.409 | 0.756 | 0.423 | 0.858 |
| | 192 | 0.310 | 0.494 | 0.361 | 0.573 | 0.360 | 0.627 | 0.353 | 0.790 | 0.349 | 0.584 | - | - | 0.350 | 0.669 | 0.456 | 0.792 | 0.410 | 0.715 | - | - | 0.396 | 0.746 | 0.444 | 0.908 |
| | 336 | 0.316 | 0.508 | 0.371 | 0.595 | 0.364 | 0.640 | 0.359 | 0.801 | 0.363 | 0.611 | - | - | 0.361 | 0.696 | 0.460 | 0.809 | 0.412 | 0.719 | - | - | 0.409 | 0.765 | 0.452 | 0.957 |
| | 720 | 0.330 | 0.534 | 0.397 | 0.645 | 0.382 | 0.676 | 0.370 | 0.860 | 0.379 | 0.642 | - | - | 0.375 | 0.750 | 0.479 | 0.863 | 0.429 | 0.758 | - | - | 0.451 | 0.830 | 0.504 | 1.055 |
| | Avg. | 0.312 | 0.499 | 0.372 | 0.593 | 0.366 | 0.635 | 0.354 | 0.803 | 0.360 | 0.603 | - | - | 0.355 | 0.692 | 0.465 | 0.824 | 0.421 | 0.742 | - | - | 0.416 | 0.774 | 0.456 | 0.945 |
| Top1 Counts. | | 55 | | 0 | | 5 | | 0 | | 4 | | 4 | | 2 | | 0 | | 5 | | 0 | | 0 | | 0 | |

Table 17: Full long-term forecasting results for DVs. We compare extensive competitive models under different prediction lengths. The input sequence length is set to 36 for the ILI dataset and 96 for the others. Avg. is averaged from all four prediction lengths. The performance is evaluated by forecasting accuracy (%). "-" denotes "out of memory".

| Models | | Mixfo. (Ours) | iTrans. (2024) | M-TCN (2024) | TimesN. (2023) | Patch. (2023) | Cross. (2023) | MICN (2023) | Light. (2023) | Dlinear (2023) | FiLM (2022) | FEDfor. (2022) | Pyrafo. (2022) |
|---|---|---|---|---|---|---|---|---|---|---|---|---|---|
| ETTm1 | 96 | 73.73 | 71.84 | 73.49 | 71.26 | 72.24 | 68.50 | 73.38 | 71.11 | 70.72 | 71.50 | 72.12 | 68.28 |
| | 192 | 70.72 | 68.80 | 71.14 | 68.33 | 69.74 | 68.00 | 70.58 | 68.58 | 68.71 | 70.16 | 67.50 | 65.95 |
| | 336 | 69.19 | 67.41 | 69.12 | 67.51 | 67.42 | 65.90 | 68.12 | 65.87 | 66.35 | 68.37 | 66.02 | 65.50 |
| | 720 | 67.81 | 65.99 | 66.94 | 65.68 | 66.58 | 63.36 | 67.47 | 64.35 | 64.91 | 67.57 | 63.30 | 62.95 |
| | Avg. | 70.36 | 68.51 | 70.17 | 68.20 | 69.00 | 66.44 | 69.89 | 67.48 | 67.67 | 69.40 | 67.24 | 65.67 |
| ETTm2 | 96 | 73.31 | 71.19 | 74.37 | 73.80 | 71.44 | 70.24 | 71.36 | 71.49 | 70.80 | 70.91 | 72.00 | 64.20 |
| | 192 | 71.25 | 70.79 | 72.65 | 73.12 | 69.59 | 70.35 | 70.20 | 67.77 | 67.76 | 69.13 | 70.12 | 64.12 |
| | 336 | 69.90 | 68.27 | 70.79 | 71.44 | 67.82 | 67.46 | 67.85 | 66.82 | 66.98 | 67.56 | 68.12 | 63.93 |
| | 720 | 68.07 | 66.85 | 69.99 | 69.08 | 65.97 | 66.37 | 67.74 | 65.82 | 64.30 | 65.60 | 66.00 | 53.98 |
| | Avg. | 70.63 | 69.28 | 71.95 | 71.86 | 68.71 | 68.61 | 69.29 | 67.98 | 67.46 | 68.30 | 69.06 | 61.56 |
| ETTh1 | 96 | 73.17 | 68.88 | 68.19 | 69.06 | 70.16 | 68.86 | 70.87 | 67.40 | 65.30 | 68.09 | 68.35 | 66.87 |
| | 192 | 68.93 | 68.20 | 67.68 | 68.36 | 69.02 | 67.07 | 69.13 | 67.04 | 65.10 | 68.74 | 66.63 | 62.62 |
| | 336 | 67.50 | 67.21 | 66.29 | 65.12 | 67.97 | 65.29 | 68.04 | 66.39 | 64.28 | 67.60 | 62.55 | 62.27 |
| | 720 | 67.28 | 63.72 | 66.53 | 54.49 | 67.01 | 66.59 | 66.41 | 66.87 | 63.58 | 65.53 | 63.74 | 59.32 |
| | Avg. | 69.22 | 67.00 | 67.17 | 64.26 | 68.54 | 66.95 | 68.61 | 66.93 | 64.57 | 67.49 | 65.32 | 62.77 |
| ETTh2 | 96 | 69.40 | 67.86 | 69.05 | 70.85 | 68.75 | 67.60 | 70.63 | 68.32 | 67.95 | 67.05 | 69.89 | 63.18 |
| | 192 | 67.01 | 65.30 | 68.85 | 67.61 | 66.49 | 65.24 | 68.02 | 66.02 | 63.87 | 65.91 | 68.33 | 55.52 |
| | 336 | 65.31 | 64.20 | 68.31 | 66.80 | 65.17 | 61.81 | 67.78 | 63.95 | 62.04 | 64.60 | 66.06 | 57.31 |
| | 720 | 65.74 | 62.27 | 68.81 | 65.87 | 63.53 | 61.78 | 68.38 | 68.22 | 60.84 | 63.25 | 64.99 | 58.82 |
| | Avg. | 66.87 | 64.91 | 68.76 | 67.78 | 65.99 | 64.11 | 68.70 | 66.63 | 63.68 | 65.20 | 67.32 | 58.71 |
| Weather | 96 | 78.74 | 76.23 | 79.37 | 78.38 | 75.69 | 79.22 | 78.48 | 77.14 | 73.91 | 75.31 | 78.83 | 77.74 |
| | 192 | 77.08 | 75.84 | 78.27 | 77.38 | 75.15 | 78.17 | 77.93 | 76.43 | 72.51 | 74.95 | 76.77 | 76.11 |
| | 336 | 76.59 | 75.55 | 78.34 | 76.18 | 75.20 | 77.77 | 75.45 | 75.11 | 72.29 | 75.52 | 74.81 | 74.92 |
| | 720 | 77.24 | 76.73 | 78.79 | 73.21 | 76.23 | 77.69 | 75.11 | 74.39 | 71.53 | 75.46 | 74.68 | 73.19 |
| | Avg. | 77.41 | 76.09 | 78.69 | 76.29 | 75.57 | 78.21 | 76.74 | 75.77 | 72.56 | 75.31 | 76.27 | 75.49 |
| Exchange | 96 | 51.16 | 48.16 | 48.21 | 49.17 | 49.17 | 48.62 | 48.42 | 48.34 | 49.91 | 49.26 | 46.44 | 49.92 |
| | 192 | 52.71 | 51.21 | 52.08 | 49.42 | 50.82 | 48.32 | 48.14 | 50.06 | 51.36 | 49.07 | 44.38 | 50.28 |
| | 336 | 52.51 | 49.90 | 48.59 | 50.67 | 50.84 | 45.24 | 44.33 | 51.42 | 51.26 | 49.55 | 42.63 | 50.53 |
| | 720 | 59.49 | 49.45 | 58.88 | 55.37 | 50.65 | 41.11 | 54.49 | 56.49 | 51.28 | 51.87 | 49.05 | 57.09 |
| | Avg. | 53.97 | 49.68 | 51.94 | 51.16 | 50.37 | 45.82 | 48.85 | 51.58 | 50.95 | 49.94 | 45.63 | 51.96 |
| ILI | 24 | 81.93 | 77.39 | 56.69 | 80.51 | 75.38 | 64.57 | 72.90 | 58.62 | 56.41 | 54.92 | 54.82 | 67.21 |
| | 36 | 84.07 | 80.36 | 55.13 | 83.22 | 79.29 | 66.19 | 71.94 | 56.94 | 55.57 | 57.17 | 61.73 | 70.51 |
| | 48 | 84.81 | 80.11 | 57.76 | 84.14 | 79.48 | 68.97 | 71.55 | 57.71 | 55.39 | 57.70 | 63.00 | 75.39 |
| | 60 | 83.37 | 78.71 | 59.47 | 81.37 | 76.99 | 69.97 | 71.50 | 61.61 | 53.23 | 58.84 | 60.56 | 74.62 |
| | Avg. | 83.55 | 79.14 | 57.26 | 82.31 | 77.79 | 67.43 | 71.97 | 58.72 | 55.15 | 57.16 | 60.03 | 71.93 |
| Electricity | 96 | 90.83 | 89.85 | 89.94 | 90.34 | 89.67 | 89.35 | 89.74 | 88.73 | 85.90 | 86.93 | 88.62 | 88.67 |
| | 192 | 90.52 | 89.80 | 89.30 | 89.69 | 89.78 | 88.36 | 89.32 | 88.61 | 85.57 | 88.49 | 89.38 | 87.93 |
| | 336 | 89.84 | 89.12 | 89.51 | 89.19 | 89.22 | 87.57 | 89.08 | 88.33 | 84.97 | 88.46 | 88.91 | 87.48 |
| | 720 | 88.74 | 87.64 | 88.36 | 88.26 | 87.86 | 86.91 | 88.01 | 88.00 | 83.76 | 87.34 | 87.89 | 87.14 |
| | Avg. | 90.40 | 89.59 | 89.58 | 89.74 | 89.56 | 88.43 | 89.38 | 88.56 | 85.48 | 87.96 | 88.97 | 88.03 |
| Traffic | 96 | 88.42 | 87.64 | 85.63 | 88.38 | 86.43 | - | 86.94 | 85.71 | 80.00 | - | 85.37 | 86.65 |
| | 192 | 88.90 | 88.72 | 87.64 | 88.98 | 87.79 | - | 88.75 | 87.80 | 81.57 | - | 88.15 | 87.21 |
| | 336 | 89.24 | 88.59 | 89.07 | 89.86 | 88.08 | - | 89.98 | 88.72 | 82.61 | - | 89.22 | 88.02 |
| | 720 | 89.71 | 89.10 | 90.48 | 90.68 | 88.46 | - | 88.91 | 89.65 | 85.30 | - | 89.59 | 89.26 |
| | Avg. | 89.07 | 88.51 | 88.21 | 89.48 | 87.69 | - | 88.65 | 87.97 | 82.37 | - | 88.08 | 87.79 |

# G   Broader Impacts

This paper copes with the general analysis of real-world mixed time series, which are under-explored in literature but frequently encountered in various practical applications, such as industrial maintenance, healthcare monitoring, and financial analysis. Since previous studies struggle to address the spatial-temporal heterogeneity problem for real-world MiTS, we present MiTSformer that fundamentally reveals and recovers the latent continuous variables for discrete variables to facilitate exploiting the intricate spatial-temporal patterns within MiTS, thereby being amenable to various analysis task. Our model achieves the state-of-the-art performance on five mainstream tasks that cover 30+ real-world datasets from diverse application domains. Therefore, the proposed model makes it promising to tackle real-world MiTS analysis tasks, which helps our society make better decisions. Our paper mainly focuses on scientific research and has no obvious negative social impact.

